# Scalable Constrained Policy Optimization for Safe Multi-agent Reinforcement Learning

**Lijun Zhang[1], Lin Li[1], Wei Wei[1]\*, Huizhong Song[1], Yaodong Yang[2], Jiye Liang[1]**

1. Key Laboratory of Computational Intelligence and Chinese Information Processing of
Ministry of Education, School of Computer and Information Technology,
Shanxi University, Taiyuan, Shanxi, China.
2. Institute for AI, Peking University, Beijing, China.

## Abstract

A challenging problem in seeking to bring multi-agent reinforcement learning (MARL) techniques into real-world applications, such as autonomous driving and drone swarms, is how to control multiple agents safely and cooperatively to accomplish tasks. Most existing safe MARL methods learn the centralized value function by introducing a global state to guide safety cooperation. However, the global coupling arising from safety constraints and the exponential growth of the state-action space size limit their applicability in instant communication or computing resource-constrained systems and larger multi-agent systems. In this paper, we develop a novel scalable and theoretically-justified multi-agent constrained policy optimization method. This method integrates the rigorous bounds of the trust region method and the bounds of the truncated advantage function to provide a new local policy optimization objective for each agent. Also, we prove that the safety constraints and the joint policy improvement can be met when each agent adopts a sequential update scheme to optimize a $\kappa$-hop policy. Furthermore, we propose a practical algorithm called Scalable MAPPO-Lagrangian (Scal-MAPPO-L). The proposed method's effectiveness is verified on a collection of benchmark tasks, and the results support our theory that decentralized training with local interactions can still improve reward performance and satisfy safe constraints.

## 1 Introduction

With the advanced and rapid developments of reinforcement learning technology, many researchers have gradually shifted their focus from virtual simulation to real-world cyber-physical applications [1, 2]. In this process, safety challenges are inevitable, especially in multi-agent safety-critical scenarios, e.g., autonomous vehicle navigation [3], power grids [4], and drone swarms [5], in which agents perform complex cooperative tasks while adhering to a variety of local and system-wide limitations or constraints. These constraints can be derived from domain-specific knowledge and are intended to prevent damage to people or other environmental elements, such as equipment and infrastructure, or to prevent the inability to accomplish specific tasks or objectives. Take multi-robot control as an example. Each running robot must not take certain actions or not visit certain states, which may imply unsafe for itself, its collaborators, or the infrastructure of its environment [6]. These widespread potential dangers exacerbate the difficulty of safety decision-making when applying MARL. Consequently, it is necessary to research the safe decision-making problem in MARL to ensure that agents can work together safely and cooperatively to accomplish tasks.

There are two main approaches concerning safe MARL techniques in the existing literature. The first type is shielded-based reactive methods [7, 8], which combine environmental dynamics and safety specification constraints to predict whether the actions chosen by agents will violate cost constraints. Nevertheless, due to the reliance on precise modeling knowledge, these methods may lead to poor performance when the accurate state transition model is unavailable. The second type formulates the safe MARL problem as a constrained Markov game, which requires agents to solve a constrained optimization problem, i.e., maximize total reward while avoiding violating cost constraints. To mention a few, several safe MARL variants, such as CMIX [9] and MAPPO-L [10], have been proposed, which learn the centralized value function to overcome policy conflicts caused by the partially observable and non-stationarity nature of the environment faced by each agent. Unfortunately, the global coupling arising from agents' safety constraints and the exponential growth of the state-action space size make the usability of these algorithms in instant communication or computing resource-constrained systems and the scalability in larger multi-agent systems become a bottleneck, limiting their applicability.

A promising approach for avoiding these shortcomings, which has received attention in recent years, is to exploit networked application-specific structures. For example, Safe Dec-PG [11] employs a primal-dual framework to find the saddle point between maximizing decoupled rewards and minimizing costs under a consensus network. However, it is worth noting that this approach still assumes each agent can access the global state and requires that the actions of all neighboring agents on the network be available. Recent research [12] proposes a scalable safe MARL approach based on the spatial decay assumption of the environment dynamics, which updates the policies of agents by the truncated gradient estimators depending on the local states and actions of the $\kappa$-hop neighboring agents. However, due to the dependence on the actions and states of its neighbors, this method necessarily involves joint training in a local area, which is still plagued by non-stationary issues. Motivated by the urgent desire for scalable learning in practical applications and the fact that meeting both safety constraints and joint policy improvement is challenging for most methods, we investigate a novel scalable safe MARL with theoretical analysis, practical algorithm, and simulation verification.

Specifically, we focus on decentralized learning settings without global observability, where each agent can only access the local state information of itself and its neighbors. Our main contributions are summarized as follows.

- We develop a novel scalable multi-agent constrained policy optimization method that eliminates dependence on the global state and other agent actions during each agent's training. Furthermore, we parameterize each agent's policy and propose a practical algorithm called Scalable MAPPO-Lagrangian (Scal-MAPPO-L).

- We quantify the maximum information loss regarding the advantage truncation based on two assumptions about the transition dynamics and policies. Then, each agent's new local policy optimization objective is provided by integrating the rigorous bounds of the trust region method and the bounds of the truncated advantage function. In addition, we prove that the safety constraints and the joint policy improvement can be guaranteed when updating the local policy with a sequential update scheme.

- Experimentally, we provide the results on several safe MARL tasks to evaluate the effectiveness of our proposed method and the sensitivity for the parameter $\kappa$. The results support our theory that decentralized training with local interactions can still improve reward performance and satisfy safe constraints.

## 2 Preliminaries

### 2.1 Constrained Markov game

Consider a safe MARL problem subject to multiple constraints, where each agent are associated with an underlying undirected graph $\mathcal{G} = (\mathcal{N}, \mathcal{E})$. Here, $\mathcal{N} = \{1, \ldots, n\}$ is the set of $n$ agents and $\mathcal{E} \subset \mathcal{N} \times \mathcal{N}$ is the set of edges. The problem can be formulated as a constrained Markov game $\langle \mathcal{N}, \mathcal{S}, \mathcal{A}, P, \boldsymbol{\rho}_0, \gamma, \boldsymbol{R}, \boldsymbol{C}, \boldsymbol{c} \rangle$. $\mathcal{S} = \times_{i \in \mathcal{N}} \mathcal{S}^i$ and $\mathcal{A} = \times_{i \in \mathcal{N}} \mathcal{A}^i$ are the state and action spaces, which are the product of local spaces; global state $\mathbf{s} = (\mathrm{s}^1, \ldots, \mathrm{s}^n)$ and joint action $\mathbf{a} = (\mathrm{a}^1, \ldots, \mathrm{a}^n)$ for any $\mathbf{s} \in \mathcal{S}$ and $\mathbf{a} \in \mathcal{A}$. $P : \mathcal{S} \times \mathcal{A} \times \mathcal{S} \longrightarrow \mathbb{R}$ is the probabilistic transition dynamics function, which

satisfies the Dobrushin condition [13] as follows:

$$W^{ij} = \sup_{z^j, z'^j, \mathbf{z}^{-j}} \left\| P^i(\cdot | z^j, \mathbf{z}^{-j}) - P^i(\cdot | z'^j, \mathbf{z}^{-j}) \right\|_1, \tag{1}$$

where $z^j = (s^j, a^j)$ and $z'^j = (s'^j, a'^j)$ represent two different state-action pairs of the agent $j$ respectively, and $\mathbf{z}^{-j}$ represents the state-action pair of the agent other than $j$. The value of $W^{ij}$ reflects the extent to which the local transition probability of agent $i$ is affected by the state and action of agent $j$. $\boldsymbol{\rho}_0$ is the initial state distribution, $\gamma \in [0, 1)$ is the discount factor. $\boldsymbol{R} : \mathcal{S} \times \mathcal{A} \longrightarrow \mathbb{R}$ is the joint reward function, $\boldsymbol{C} = \{C^i_j\}^{i \in \mathcal{N}}_{1 \leq j \leq m^i}$ is the sets of cost functions (every agent $i$ has $m^i$ cost functions) of the form $C^i_j : \mathcal{S}^i \times \mathcal{A}^i \longrightarrow \mathbb{R}$, and finally the set of corresponding cost values is given by $\boldsymbol{c} = \{c^i_j\}^{i \in \mathcal{N}}_{1 \leq j \leq m^i}$.

At each timestep $t$, every agent $i$ is in a state $s^i_t$, and takes an action $a^i_t$ according to its policy $\pi^i = (a^i | s^i_t)$. Together with other agents actions, it gives a joint action $\mathbf{a}_t = (a^1_t, \ldots, a^n_t)$ and the joint policy $\boldsymbol{\pi} = \prod^n_{i=1} \pi^i(a^i | s^i_t)$. The agents receive the reward $\boldsymbol{R}(\mathbf{s}_t, \mathbf{a}_t)$, meanwhile each agent $i$ pays the costs $C^i_j(s^i_t, a^i_t), \forall\, j = 1, \ldots, m^i$, and all agents have a joint goal, i.e., maximizing the expected total reward of

$$J(\boldsymbol{\pi}) \triangleq \mathbb{E}_{\mathbf{s}_0 \sim \boldsymbol{\rho}_0, \mathbf{a}_{0:\infty} \sim \boldsymbol{\pi}} \left[ \sum^{\infty}_{t=0} \gamma^t \boldsymbol{R}(\mathbf{s}_t, \mathbf{a}_t) \right], \tag{2}$$

meanwhile satisfying every agent $i$'s safety constraints, written as

$$J^i_j(\boldsymbol{\pi}) \triangleq \mathbb{E}_{\mathbf{s}_0 \sim \boldsymbol{\rho}_0, \mathbf{a}_{0:\infty} \sim \boldsymbol{\pi}} \left[ \sum^{\infty}_{t=0} \gamma^t C^i_j(\mathbf{s}_t, a^i_t) \right] \leq c^i_j, \forall\, j = 1, \ldots, m^i. \tag{3}$$

## 2.2 Spatial correlation decay

Exponential decay property [13, 14], also known as spatial correlation decay, is a powerful property associated with local interactions, which says that the impact of agents on each other decays exponentially in their graph distance. More information about spatial correlation decay is presented in Appendix B.1. Here, inspired by [15], we make the following two assumptions for the spatial correlation of the transition dynamics and policies. We use the notation $\pi^i(\cdot | s_{\mathcal{N}^i_\kappa})$ for $\kappa$-hop policies, where $s_{\mathcal{N}^i_\kappa}$ represents the state of agent $i$'s $\kappa$-hop neighbors. It may be replaced with $\pi^i_\kappa$ for simplicity when it is clear from context.

**Assumption 2.1.** (Spatial Decay of Correlation for the Dynamics) Assume that there exist $\beta > 0$ in (1), for any agents $i, j \in \mathcal{N}$, such that

$$\max_{i \in \mathcal{N}} \sum_{j \in \mathcal{N}} e^{\beta d(i,j)} W^{ij} \leq \zeta, \tag{4}$$

where $d(i, j)$ represents the distance between agent $i$ and agent $j$, and $\zeta \in [0, 2/\gamma)$ is a constant.

**Assumption 2.2.** (Spatial Decay of Correlation for the Policies) Assume that there exist $\xi, \beta \geq 0$ such that for any agent $i \in \mathcal{N}$, $s_{\mathcal{N}^i_\kappa} \in \mathcal{S}_{\mathcal{N}^i_\kappa}$, $\mathbf{s}_{\mathcal{N}^{-i}_\kappa}, \mathbf{s}'_{\mathcal{N}^{-i}_\kappa} \in \mathcal{S}_{\mathcal{N}^{-i}_\kappa}$, one have

$$\sup_{s_{\mathcal{N}^i_\kappa}, \mathbf{s}_{\mathcal{N}^{-i}_\kappa}, \mathbf{s}'_{\mathcal{N}^{-i}_\kappa}} \left| \pi^i \left( \cdot | s_{\mathcal{N}^i_\kappa}, \mathbf{s}_{\mathcal{N}^{-i}_\kappa} \right) - \pi^i \left( \cdot | s_{\mathcal{N}^i_\kappa}, \mathbf{s}'_{\mathcal{N}^{-i}_\kappa} \right) \right| \leq \xi e^{-\beta \kappa}. \tag{5}$$

Assumption 2.2 reveals how much information is lost compared with access to the global state and allows us to consider a policy class with the necessary properties for the optimal policy under Assumption 2.1. More information is stated in Appendix B.2.

# 3 Scalable constrained policy optimization

This section develops a novel scalable and theoretically-justified multi-agent constrained policy optimization method and proposes a practical algorithm, i.e., Scal-MAPPO-L, by parameterizing each agent's policy. Specifically, we first quantify the maximum information loss regarding the

advantage truncation based on the spatial correlation decay property of the transition dynamics and policies. Then, the rigorous bounds of the trust region method and the bounds of the truncated advantage function are integrated to provide a new local policy optimization objective for each agent. Further, we prove that the safety constraints and the joint policy improvement can be guaranteed when updating the local police with a sequential update scheme, in which the policy update only depends on its action and the state of its $\kappa$-hop neighbors for each agent.

## 3.1 Truncated advantage function estimator

For a standard safe MARL, the state-action value function (the definition can be seen in Appendix C.1) and advantage function of agent $i$ yield that

$$Q_{\boldsymbol{\pi}}^i(\mathbf{s}, \mathrm{a}^i) = \mathbb{E}_{\mathbf{a}^{-i} \sim \boldsymbol{\pi}^{-i}} Q_{\boldsymbol{\pi}}^i(\mathbf{s}, \mathbf{a}^{-i}, \mathrm{a}^i), \tag{6}$$

$$A_{\boldsymbol{\pi}}^i(\mathbf{s}, \mathrm{a}^j, \mathrm{a}^i) = Q_{\boldsymbol{\pi}}^{j,i}(\mathbf{s}, \mathrm{a}^j, \mathrm{a}^i) - Q_{\boldsymbol{\pi}}^j(\mathbf{s}, \mathrm{a}^j). \tag{7}$$

where $\mathbf{s}$ represents the global state, $\mathbf{a}^{-i}$ represents the actions of all other agents, and $Q_{\boldsymbol{\pi}}^{j,i}(\mathbf{s}, \mathrm{a}^j, \mathrm{a}^i)$ represents the state-action value function of agent $i$ and agent $j$. Then, updating agents' policies with a sequential update scheme [16], the multi-agent joint advantage function $\boldsymbol{A}_{\boldsymbol{\pi}}(\mathbf{s}, \mathbf{a})$ can be written as a sum of sequentially unfolding multi-agent advantages of individual agents, as stated by the following lemma.

**Lemma 3.1.** *(Multi-agent advantage decomposition). For any action $\mathrm{a}^i$, $i \in \mathcal{N}$, and the state $\mathbf{s} \in \mathcal{S}$, the following identity holds*

$$\boldsymbol{A}_{\boldsymbol{\pi}}(\mathbf{s}, \mathbf{a}) = \sum_{i=1}^{n} A_{\boldsymbol{\pi}}^i(\mathbf{s}, \mathbf{a}^{-i}, \mathrm{a}^i). \tag{8}$$

Similar result to Lemma 3.1 can be seen in [10], and the proof is reported in Appendix C.2. Specifically, based on the multi-agent advantage decomposition in Lemma 3.1, the "surrogate" return is given as follows.

**Definition 3.2.** Let $\boldsymbol{\pi}$ be a joint policy, $\bar{\boldsymbol{\pi}}^{1:i-1}$ be some other joint policy of agents $1 : i - 1$, and $\hat{\pi}^i$ be a policy of agent $i$. Then, the surrogate return can be defined as

$$L_{\boldsymbol{\pi}}^{1:i}\left(\bar{\boldsymbol{\pi}}^{1:i-1}, \hat{\pi}^i\right) \triangleq \mathbb{E}_{\mathbf{s} \sim \rho_{\boldsymbol{\pi}}, \mathbf{a}^{1:i-1} \sim \bar{\boldsymbol{\pi}}^{1:i-1}, \mathrm{a}^i \sim \hat{\pi}^i}\left[A_{\boldsymbol{\pi}}^i\left(\mathbf{s}, \mathbf{a}^{1:i-1}, \mathrm{a}^i\right)\right]. \tag{9}$$

Building on Lemma 3.1 and Definition 3.2, one can obtain

$$L_{\boldsymbol{\pi}}^{1:i}\left(\bar{\boldsymbol{\pi}}^{1:i-1}, \bar{\pi}^i\right) = \mathbb{E}_{\mathbf{s} \sim \rho_{\boldsymbol{\pi}}, \mathbf{a}^{1:i} \sim \bar{\boldsymbol{\pi}}^{1:i}}\left[\sum_{h=1}^{i} A_{\boldsymbol{\pi}}^h\left(\mathbf{s}, \mathbf{a}^{1:h-1}, \mathrm{a}^h\right)\right]. \tag{10}$$

Further, recalling Assumption 2.1 and Assumption 2.2, we can quantify the maximum information loss regarding the advantage function as stated by the following proposition.

**Proposition 3.3.** *For any agent $i \in \mathcal{N}$, let the parameters $(\eta, \phi) = \left(\frac{\xi \gamma \zeta}{1 - \gamma \zeta}, e^{-\beta}\right)$. If Assumption 2.1 and Assumption 2.2 hold, for any $\mathrm{z}_{\mathcal{N}_\kappa^i} = \left(\mathrm{s}_{\mathcal{N}_\kappa^i}, \mathrm{a}_{\mathcal{N}_\kappa^i}\right) \in \mathcal{S}_{\mathcal{N}_\kappa^i} \times \mathcal{A}_{\mathcal{N}_\kappa^i}$, the exponential decay property of the advantage function holds, i.e., we have*

$$\sup_{\mathrm{z}_{\mathcal{N}_\kappa^i}, \mathbf{z}_{\mathcal{N}_\kappa^{-i}}, \mathbf{z}'_{\mathcal{N}_\kappa^{-i}}} \left|A^i\left(\mathrm{z}_{\mathcal{N}_\kappa^i}, \mathbf{z}_{\mathcal{N}_\kappa^{-i}}\right) - A^i\left(\mathrm{z}_{\mathcal{N}_\kappa^i}, \mathbf{z}'_{\mathcal{N}_\kappa^{-i}}\right)\right| \leq \eta \phi^\kappa. \tag{11}$$

Proposition 3.3 shows that when the transition dynamics and policies correlation satisfy the exponential correlation decay property, the advantage functions also have exponential decay dependence on the states and actions of the more distant agents. The proof of Proposition 3.3 is reported in Appendix C.3. In addition, based on this proposition, we can obtain the following corollary.

**Corollary 3.4.** *For any agent $i \in \mathcal{N}$, let the parameters $(\eta', \phi) = \left(\frac{M^i \xi}{1 - \gamma} + \frac{(2 + \xi) \gamma \zeta}{1 - \gamma \zeta}, e^{-\beta}\right)$, $M^i$ is a constant. If Proposition 3.3 holds, the exponential decay property of the surrogate return holds, i.e., we have*

$$\left|L_{\boldsymbol{\pi}}^{1:i}\left(\bar{\boldsymbol{\pi}}^{1:i-1}, \bar{\pi}^i\right) - L_{\pi_\kappa^i}^i\left(\bar{\pi}_\kappa^i\right)\right| \leq \eta' \phi^\kappa. \tag{12}$$

The proofs of Corollary 3.4 is reported in Appendix C.4.

Corollary 3.4 shows that the approximation error of $L^i_{\pi^i_\kappa}\left(\bar{\pi}^i_\kappa\right)$ decreases exponentially with $\kappa$ when the truncated advantage functions are bounded. The main advantage of using the estimator $L^i_{\pi^i_\kappa}\left(\bar{\pi}^i_\kappa\right)$ lies in that every agent $i$ only needs to know the action and state of its $\kappa$-hop neighbors, which can significantly reduce the communication burden and expand its application scenarios.

## 3.2 Scalable constrained policy optimization

With the Definition 3.2, we see that Lemma 3.1 allows for decomposing the joint surrogate return $L_{\boldsymbol{\pi}}\left(\bar{\boldsymbol{\pi}}\right) \triangleq \mathbb{E}_{\mathbf{s}\sim\rho_{\boldsymbol{\pi}},\mathbf{a}\sim\bar{\boldsymbol{\pi}}}\left[A_{\boldsymbol{\pi}}\left(\mathbf{s},\mathbf{a}\right)\right]$ into a sum over surrogates of $L^{1:i}_{\boldsymbol{\pi}}\left(\bar{\boldsymbol{\pi}}^{1:i-1},\hat{\pi}^i\right)$. Then, combining the rigorous bounds of the trust region method [17] and the bounds of the truncated advantage function, we can obtain the following proposition.

**Proposition 3.5.** *Let $\boldsymbol{\pi}$ and $\bar{\boldsymbol{\pi}}$ be joint policies. Let each agent $i \in \mathcal{N}$ sequentially solves the following optimization problem:*

$$\bar{\pi}^i_\kappa = \arg\max_{\hat{\pi}^i_\kappa}\left(L^i_{\pi^i_\kappa}\left(\hat{\pi}^i_\kappa\right) - \eta'\phi^\kappa - \nu^i_\kappa D^{\max}_{\mathrm{KL}}\left(\pi^i_\kappa|\hat{\pi}^i_\kappa\right)\right), \tag{13}$$

*where $(\eta',\phi) = \left(\frac{M^i\xi}{1-\gamma} + \frac{(2+\xi)\gamma\zeta}{1-\gamma\zeta}, e^{-\beta}\right)$, $\nu^i_\kappa = \frac{2\gamma\max_{\mathbf{s}_{\mathcal{N}^i_\kappa},\mathbf{a}^i}\left|A^i_{\pi^i_\kappa}(\mathbf{s}_{\mathcal{N}^i_\kappa},\mathbf{a}^i)\right|}{(1-\gamma)^2}$, and $D^{\max}_{\mathrm{KL}}\left(\pi^i_\kappa|\hat{\pi}^i_\kappa\right) = \max_{\mathbf{s}_{\mathcal{N}^i_\kappa}} D_{\mathrm{KL}}\left(\pi^i(\cdot\mid\mathbf{s}_{\mathcal{N}^i_\kappa}),\hat{\pi}^i(\cdot\mid\mathbf{s}_{\mathcal{N}^i_\kappa})\right)$, then the resulting joint policy $\bar{\boldsymbol{\pi}}$ will improve the expected return, i.e.,*

$$J\left(\bar{\boldsymbol{\pi}}\right) - J\left(\boldsymbol{\pi}\right) \geq \sum_{i=1}^{N}\left(L^i_{\pi^i_\kappa}\left(\hat{\pi}^i_\kappa\right) - \eta'\phi^\kappa - \nu^i_\kappa D^{\max}_{\mathrm{KL}}\left(\pi^i_\kappa|\hat{\pi}^i_\kappa\right)\right). \tag{14}$$

The proof of Proposition 3.5 is reported in Appendix C.5. Similarly, by generalizing the result about the surrogate return in Equation (12), we can derive how the expected costs change when the agents update their policies. Specifically, we provide the following corollary.

**Corollary 3.6.** *Let $\boldsymbol{\pi}$ and $\bar{\boldsymbol{\pi}}$ be joint policies. For any agent $i \in \mathcal{N}$ and its cost index $j \in \{1,\ldots,m^i\}$, the following inequality holds*

$$J^i_j(\bar{\boldsymbol{\pi}}) \leq J^i_j(\boldsymbol{\pi}) + L^i_{j,\pi^i_\kappa}\left(\bar{\pi}^i_\kappa\right) + \eta''\phi^\kappa + \nu^i_{j,\kappa}\sum_{h=1}^{i-1} D^{\max}_{\mathrm{KL}}\left(\pi^h_\kappa,\bar{\pi}^h_\kappa\right), \tag{15}$$

*where $L^i_{j,\pi^i_\kappa}\left(\bar{\pi}^i_\kappa\right) = \mathbb{E}_{\mathbf{s}_{\mathcal{N}^i_\kappa}\sim\rho_{\pi^i_\kappa},\mathbf{a}^i\sim\bar{\pi}^i_\kappa}\left[A^i_{j,\pi^i_\kappa}\left(\mathbf{s}_{\mathcal{N}^i_\kappa},\mathbf{a}^i\right)\right]$, $\nu^i_{j,\kappa} = \frac{2\gamma\max_{\mathbf{s}_{\mathcal{N}^i_\kappa},\mathbf{a}^i}\left|A^i_{j,\pi^i_\kappa}\left(\mathbf{s}_{\mathcal{N}^i_\kappa},\mathbf{a}^i\right)\right|}{(1-\gamma)^2}$, $(\eta'',\phi) = \left(\frac{M_j\xi}{1-\gamma} + \frac{(2+\xi)\gamma\zeta}{1-\gamma\zeta}, e^{-\beta}\right)$, and $M_j$ is a constant.*

The proofs of Corollary 3.6 is reported in Appendix C.6.

From (14), we can derive that the lower bound for the difference between the new joint policy $\bar{\boldsymbol{\pi}}$ and the old joint policy $\boldsymbol{\pi}$ in terms of expected return can be decomposed into a cumulative sum of local surrogate TRPO policy objectives. From (15), we can derive the upper bound for the new joint policy $\bar{\boldsymbol{\pi}}$, which can be used to restrict agents only to choose safe actions. Therefore, we use the objective, i.e., maximize the lower bound for the reward performance and minimize the upper bound for the safety constraints with a proper update size, as a surrogate for each agent. Then, we can obtaine the following theorem.

**Theorem 3.7.** *The joint policy $\boldsymbol{\pi}$ has the monotonic improvement property, $J\left(\bar{\boldsymbol{\pi}}\right) \geq J\left(\boldsymbol{\pi}\right)$, as well as it satisfies the safety constraints, $J^i_j\left(\bar{\boldsymbol{\pi}}\right) \leq c^i_j$, for any agent $i \in \mathcal{N}$ and its cost index $j \in \{1,\ldots,m^i\}$, when the policy is updated by following a sequential update scheme, that is, each agent sequentially solves the following optimization problem:*

$$\bar{\pi}^i_\kappa = \arg\max_{\hat{\pi}^i_\kappa\in\bar{\Pi}^i_\kappa}\left(L^i_{\pi^i_\kappa}\left(\hat{\pi}^i_\kappa\right) - \eta'\phi^\kappa - \nu^i_\kappa D^{\max}_{\mathrm{KL}}\left(\pi^i_\kappa|\hat{\pi}^i_\kappa\right)\right),$$

$$s.t. \left\{\hat{\pi}^i_\kappa\in\bar{\Pi}^i_\kappa\mid D^{\max}_{\mathrm{KL}}\left(\pi^i_\kappa,\hat{\pi}^i_\kappa\right)\leq\delta^i_\kappa, and\right. \tag{16}$$

$$\left. J^i_j\left(\pi_\kappa\right) + L^i_{j,\pi^i_\kappa}\left(\hat{\pi}^i_\kappa\right) + \eta''\phi^\kappa + \nu^i_{j,\kappa}D^{\max}_{\mathrm{KL}}\left(\pi^i_\kappa,\hat{\pi}^i_\kappa\right)\leq c^i_j - \nu^i_{j,\kappa}\sum_{h=1}^{i-1}D^{\max}_{\mathrm{KL}}\left(\pi^h_\kappa,\hat{\pi}^h_\kappa\right)\right\},$$

$$\text{where} \quad \delta_\kappa^i \;=\; \min\left\{\min_{h\le i-1}\min_{1\le j\le m^h}\frac{\Xi_j^h - L_{j,\pi_\kappa^h}^h\left(\bar\pi_\kappa^h\right) - \eta''\phi^\kappa}{\nu_{j,\kappa}^i}, \min_{h\ge i+1}\min_{1\le j\le m^h}\frac{\Xi_j^h}{\nu_{j,\kappa}^i}\right\},$$

$$\nu_\kappa^i = \frac{2\gamma\max_{\mathrm{s}_{\mathcal{N}_\kappa^i},\mathrm{a}^i}\left|A_{\pi_\kappa^i}^i\left(\mathrm{s}_{\mathcal{N}_\kappa^i},\mathrm{a}^i\right)\right|}{(1-\gamma)^2}, \nu_{j,\kappa}^i = \frac{2\gamma\max_{\mathrm{s}_{\mathcal{N}_\kappa^i},\mathrm{a}^i}\left|A_{j,\pi_\kappa^i}^i\left(\mathrm{s}_{\mathcal{N}_\kappa^i},\mathrm{a}^i\right)\right|}{(1-\gamma)^2}, \quad (\eta',\phi) = \left(\frac{M^i\xi}{1-\gamma}+\frac{(2+\xi)\gamma\zeta}{1-\gamma\zeta},e^{-\beta}\right), (\eta'',\phi) = \left(\frac{M_j\xi}{1-\gamma}+\frac{(2+\xi)\gamma\zeta}{1-\gamma\zeta},e^{-\beta}\right), \quad \Xi_j^h = c_j^h - J_j^h\left(\pi_\kappa^h\right) - \nu_{j,\kappa}^h\sum_{l=1}^{i-1}D_{\mathrm{KL}}^{\max}\left(\pi_\kappa^l,\hat\pi_\kappa^l\right).$$

The proof of Theorem 3.7 is reported in Appendix C.7. It assures that if one follows (16) to update policies, agents will not only explore safe policies independently; meanwhile, every new policy will be guaranteed to result in performance improvement. It is worth mentioning that these two properties hold only under the condition that the only policy update restriction, i.e., $\bar\pi_\kappa^i \in \bar\Pi_\kappa^i$, is satisfied; this is due to the KL-penalty term in every agent's objective, i.e., $\nu_\kappa^i D_{\mathrm{KL}}^{\max}\left(\pi_\kappa^i,\bar\pi_\kappa^i\right)$, as well as the constraints on cost surrogates.

### 3.3 Algorithm

In this section, we focus on how to practically implement policy updates in Theorem 3.7 for each agent. Specifically, we parameterize each local policy $\pi_{\theta_\kappa^i}^i$ by a neural network with parameter $\theta_\kappa^i$. At each policy update, every agent $i$ maximizes its surrogate return subject to surrogate cost constraints and a form of expected KL-divergence constraint $\widetilde{D}_{\mathrm{KL}}\left(\pi_\kappa^i,\bar\pi_\kappa^i\right) \le \delta_\kappa^i$, which avoids computing KL-divergence at every state. Then, we introduce a scalar variable $\lambda^i$ for any agent $i \in \mathcal{N}$ and convert the constrained optimization problem from (16) into a min-max optimization problem with Lagrangian multipliers by subsuming the cost constraints. As such, the new optimization problem for any agent $i \in \mathcal{N}$ is as follows:

$$\max_{\theta_\kappa^i}\min_{\lambda_{1:m^i}^i\ge 0}\left[\mathbb{E}_{\mathrm{s}_{\mathcal{N}_\kappa^i}\sim\rho_{\pi_{\theta_\kappa^i}^i},\mathrm{a}^i\sim\pi_{\theta_\kappa^i}^i}\left[A_{\pi_{\theta_\kappa^i}^i}^i\left(\mathrm{s}_{\mathcal{N}_\kappa^i},\mathrm{a}^i\right)\right]\right.$$
$$\left.-\sum_{u=1}^{m^i}\lambda_u^i\left(\mathbb{E}_{\mathrm{s}_{\mathcal{N}_\kappa^i}\sim\rho_{\pi_{\theta_\kappa^i}^i},\mathrm{a}^i\sim\pi_{\theta_\kappa^i}^i}\left[A_{u,\pi_{\theta_\kappa^i}^i}^i\left(\mathrm{s}_{\mathcal{N}_\kappa^i},\mathrm{a}^i\right)\right]+d_u^i\right)\right], \tag{17}$$
$$\text{s.t.}\,\widetilde{D}_{\mathrm{KL}}\left(\pi_{\theta_\kappa^i}^i,\bar\pi_{\theta_\kappa^i}^i\right) \le \delta_\kappa^i.$$

where $\lambda_{1:m^i}^i$ is a scalar variable, $\theta_\kappa^i$ is a parameter of neural network, and $d_u^i$ is the cost-constraining value for agent $i$.

Further, denoting

$$A_{\pi_{\theta_\kappa^i}^i}^{i,(\lambda)}\left(\mathrm{s}_{\mathcal{N}_\kappa^i},\mathrm{a}^i\right) = A_{\pi_{\theta_\kappa^i}^i}^i\left(\mathrm{s}_{\mathcal{N}_\kappa^i},\mathrm{a}^i\right) - \sum_{u=1}^{m^i}\lambda_u^i\left(A_{u,\pi_{\theta_\kappa^i}^i}^i\left(\mathrm{s}_{\mathcal{N}_\kappa^i},\mathrm{a}^i\right)+d_u^i\right), \tag{18}$$

then the optimization problem in (17) can be rewritten as

$$\max_{\theta_\kappa^i}\min_{\lambda_{1:m^i}^i\ge 0}\left[\mathbb{E}_{\mathrm{s}_{\mathcal{N}_\kappa^i}\sim\rho_{\pi_{\theta_\kappa^i}^i},\mathrm{a}^i\sim\pi_{\theta_\kappa^i}^i}\left[A_{\pi_{\theta_\kappa^i}^i}^{i,(\lambda)}\left(\mathrm{s}_{\mathcal{N}_\kappa^i},\mathrm{a}^i\right)\right]\right], \text{ s.t. } \widetilde{D}_{\mathrm{KL}}\left(\pi_{\theta_k^i}^i,\bar\pi_{\theta_\kappa^i}^i\right) \le \delta_\kappa^i. \tag{19}$$

To alleviate the complications caused by computing the KL-divergence constraint, we simplify it by adopting the PPO-clip objective [18], i.e., replacing the KL-divergence constraint with the clip operator and updating the policy parameter with first-order methods. The final optimization problem takes the form

$$\max_{\theta_\kappa^i}\min_{\lambda_{1:m^i}^i\ge 0}\mathbb{E}_{\mathrm{s}_{\mathcal{N}_\kappa^i}\sim\rho_{\pi_{\theta_\kappa^i}^i},\mathrm{a}^i\sim\pi_{\theta_\kappa^i}^i}\left[\min\left(\frac{\bar\pi_{\theta_\kappa^i}^i}{\pi_{\theta_\kappa^i}^i}A_{\pi_{\theta_\kappa^i}^i}^{i,(\lambda)}\left(\mathrm{s}_{\mathcal{N}_\kappa^i},\mathrm{a}^i\right),\left(\frac{\bar\pi_{\theta_\kappa^i}^i}{\pi_{\theta_\kappa^i}^i},1\pm\epsilon\right)A_{\pi_{\theta_\kappa^i}^i}^{i,(\lambda)}\left(\mathrm{s}_{\mathcal{N}_\kappa^i},\mathrm{a}^i\right)\right)\right],$$
$$\tag{20}$$

where the clip operator replaces the policy ratio with $1 + \epsilon$, or $1 - \epsilon$, depending on whether its value is below or above the threshold interval. As such, agent $i$ can learn within its trust region by updating

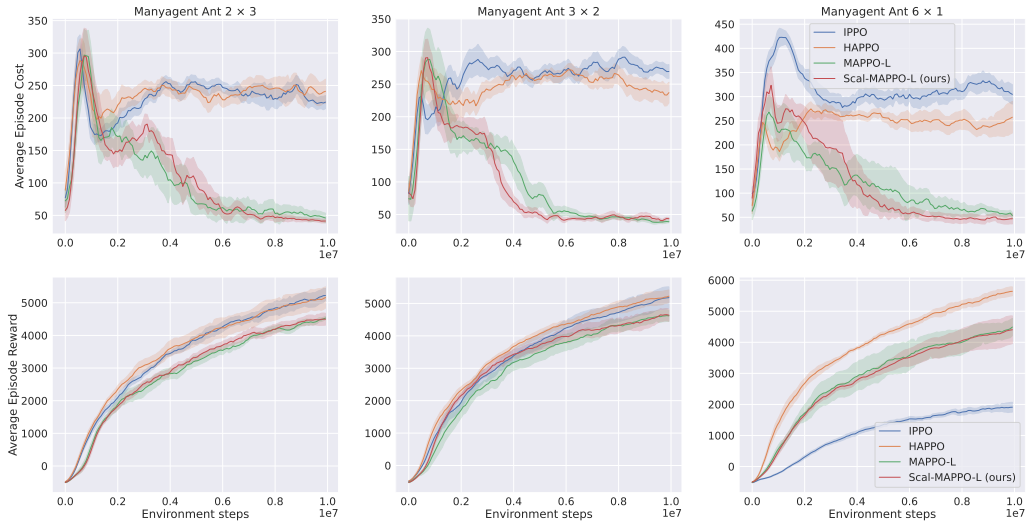

Figure 1: Performance comparisons in terms of cost and reward on three Safe ManyAgent Ant tasks. Each column subfigure represents a different task, and we plot the cost curves (the lower the better) in the upper row and the reward curves (the higher the better) in the bottom row for each task.

$\theta_\kappa^i$ to maximize Equation (20), which only depends on its action and the state of its $\kappa$-hop neighbors and can be computed analytically.

To summarize, we give a procedure for each agent $i$, name Scalable MAPPO-Lagrangian (Scal-MAPPO-L), and provide its pseudocode (Algorithm 1) in Appendix C.8. The algorithm has a simple idea that each agent independently optimizes the surrogate objective (20), which only depends on its action and the state of its $\kappa$-hop neighbors for each agent. In the actual execution, some approximations of the surrogate objective are employed, the same as the MAPPO-L [10]. Most of these approximations are traditional practices in RL, yet they may make it impossible for the practical algorithm to rigorously maintain the theoretical guarantees in Theorem 3.7.

## 4 Experiments

In this section, we evaluate our method via several numerical experiments. Our experiments aim to answer the following questions: First, how does the cost and reward performance of Scal-MAPPO-L compare with existing methods on challenging multi-agent safe tasks? Second, how does the different $\kappa$ affect the performance of Scal-MAPPO-L, and could the advantage truncation effectively alleviate computational load?

### 4.1 Experimental setup

Safe MAMuJoCo [10] is an extension of MAMuJoCo [19], which preserves the agents, physics simulator, background environment, and reward function and comes with obstacles, like walls or pitfalls. To answer the first question, we compare our method against the other PPO family algorithms, i.e., IPPO [20], HAPPO [16], and MAPPO-L [10] and choose three games from Safe MAMuJoCo: Safe ManyAgent Ant task with 2 agents ($2 \times 3$), 3 agents ($3 \times 2$) and 6 agents ($6 \times 1$) to evaluate their performance. Concerning the second question, we choose three games with different tasks and agent numbers from Safe MAMuJoCo: Safe ManyAgent Ant task with 6 agents ($6 \times 1$), Safe Ant task with 8 agents ($8 \times 1$), and Safe Coupled HalfCheetah task with 12 agents ($12 \times 1$). We train Scal-MAPPO-L with the same network architecture and hyperparameters as the original MAPPO-L implementation. All reported results are averaged over three or more random seeds, and the curves are smooth over time.

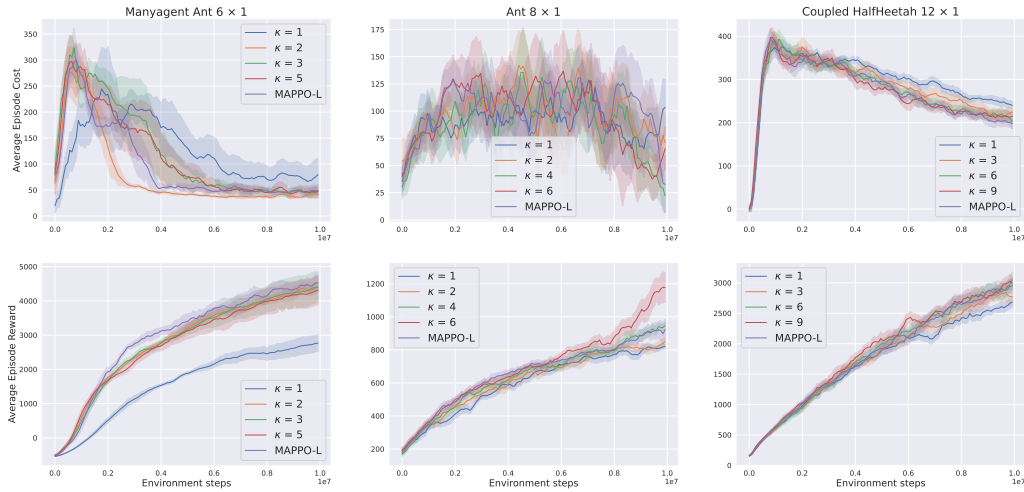

Figure 2: Performance comparisons in terms of cost and reward on Safe ManyAgent Ant task, Safe Ant task, and Safe Coupled HalfCheetah task. In each task, the performance of Scal-MAPPO-L with different $\kappa$ and MAPPO-L are demonstrated.

## 4.2 Results

**Comparisons with baselines:** Figure 1 shows the cost and reward performance of Scal-MAPPO-L and other PPO family algorithms on three Safe ManyAgent Ant tasks, where each agent in Scal-MAPPO-L is set to access the state of about half of the agents by adjusting the value of $\kappa$. Specifically, $\kappa = 1$ in Safe ManyAgent Ant ($2 \times 3$), $\kappa = 2$ in Safe ManyAgent Ant ($3 \times 2$), and $\kappa = 3$ in Safe ManyAgent Ant ($6 \times 1$). From Figure 1, we can see that compared to IPPO and HAPPO, on all three tasks, both Scal-MAPPO-L and MAPPO-L have fewer constraint violations and good performance (in terms of reward), i.e., they keep their explorations within the feasible policy space and quickly learn to satisfy safety constraints, which show that the safe learning algorithm is effective. Moreover, it should be further pointed out that Scal-MAPPO-L only accesses half of the state information on all tasks; it exhibits almost identical performance and constraint violations with MAPPO-L (which accesses the global state). This means that the sensitivity of each agent to the states and actions perturbations of distant agents is minimal, and Scal-MAPPO-L is effective. More experimental results are in Appendix D.

**Performance with different $\kappa$:** Figure 2 shows the performance of Scal-MAPPO-L in different environments with varying values of $\kappa$, where MAPPO-L accesses the global state. We have noticed that the algorithm's performance is consistently the lowest, and the cost is nearly the highest when $\kappa = 1$. However, when the truncation with $\kappa >= 3$, i.e., each agent has access to the states of at least two neighbors, we can observe that the performance of Scal-MAPPO-L improves considerably and can approach or even outperform MAPPO-L in some environments, such as $\kappa = 6$ in the Safe Ant task ($8 \times 1$). This may be due to the fact that the impact of far-away agents' states and actions on the agent's decision is almost negligible in many cases. However, for algorithms with global communication, such as MAPPO-L, the difficulty of extracting useful information from many messages may lead to lower performance. Overall, these results underscore the efficiency of Scal-MAPPO-L since it employs a smaller communication radius that can significantly reduce the computation.

## 5 Related work

### 5.1 Safe RL

Safety is one of the bottlenecks preventing RL use in real-life applications, such as physical robotics [21], medical applications [22] and autonomous driving [23]. It has become a research hotspot in recent years and a growing number of safe RL approaches, such as primal-dual methods [24], formal methods [25], Lyapunov methods [26], Gaussian processes methods [27], and safety-augmented

methods [28], have been developed. However, when it comes to multi-agent systems, a great challenge is exacerbated by policy conflicts caused by multiple agents interacting within a shared environment and learning simultaneously. In other words, each agent has to not only satisfy its safety constraints but also consider the conflicts between its safety constraints and maximization reward as well as the safety constraints of others so that their joint behaviors have a safety guarantee. In order to address the above issue, CMIX [9] and MAPPO-L [10] have been proposed with the in-depth study of MARL. These algorithms follow the centralized training and decentralized execution (CTDE) framework [29, 30, 16], which learns the centralized value function by introducing the global state. Unfortunately, the global coupling arising from agents' safety constraints and the exponential growth of the state-action space size make the usability in communication or computing resource-constrained systems and the scalability of these algorithms in larger multi-agent systems become a bottleneck, limiting their applicability. Recent works [11, 12] have provided some theoretical results to avoid these shortcomings. However, most of these methods fail to ensure both safety guarantee and joint policy improvement under a decentralized learning framework under a decentralized learning framework, which motivates us to investigate a new scalable and theoretically-justified safe MARL method.

## 5.2 Centralized training

In cooperative MARL settings, the training of agents can be broadly divided into two paradigms, namely centralized and decentralized [31]. The centralized training paradigm describes agent policies updated based on mutual information, which can be further differentiated into the centralized and decentralized execution framework. Centralized training and centralized execution (CTCE) utilize the centralized evaluator and executor to learn the joint policy of all agents [32, 18]. The obvious flaw is that its applicability is limited because its implementation requires the premise that instantaneous and unconstrained information exchange between agents. Recently, centralized training and decentralized execution (CTDE) has become the most popular framework [30, 20, 16, 10], since the fact that it addresses the non-stationarity issue with the centralized value function, and removes the dependency on global state and actions during execution. Many experiment results demonstrate state-of-the-art performance on challenging tasks, such as unit micromanagement in StarCraft II [33]. However, although this framework does not require agents to access the global state during execution, the reliance on the global state only during training still poses a significant barrier to real-world applications, especially in scenarios where communication and computational resources are constrained [34, 35].

## 5.3 Decentralized training

In a decentralized learning paradigm, each agent learns independently and accessses local observations rather than the global state; the idea is direct, comprehensible, and easy to realize in practice [36, 34]. There are two mainline research approaches concerning decentralized learning in the existing literature. One line of research pursues fully decentralized learning, such as independent Q-learning (IQL) [37, 38] and independent actor-critic (IAC) [39, 20], which make agents directly execute the single-agent Q-learning or actor-critic algorithm individually. Another line of research allows agents to establish rational local communication networks, such as setting certain distance or neighbor graphs [40, 41], which is also known as networked MARL. Communication networks expand agents' perceptual capabilities and mitigate, to some extent, the decision conflicts or errors caused by partial observability. However, it is worth noting that each agent's decision violates the stationary condition of the Markov Decision Process (MDP) in both lines of research, even though they achieve good experimental results on a collection of benchmark tasks. It poses a significant challenge to the convergence analysis of algorithms in the short term. Recently, motivated by good experiment performance, some studies have tried to provide theoretical support for these phenomena. To mention a few, Qu et al. [42] introduced the spatial correlation decay property into the field of MARL and carried out a series of fundamental results [15, 43, 12], which broadened the research avenues of scalable MARL. However, all of these studies mainly focus on (natural) policy gradient methods with average rewards or general utilities and have not yet been combined with trust region methods, which rigorously enable RL agents to learn monotonically improving policies. Furthermore, only recent research [12] considers both safety and scalability for MARL. Our results build upon the scalable MARL family of works [42, 15, 43, 12] and PPO-based (TRPO-based) MARL family of works [16, 10].

# 6 Conclusion

Safety is a tremendous challenge for MARL when applied to real-world scenarios. In this paper, we quantize the approximation errors arising from policy implementation and advantage truncation and then derive a novel lower bound for joint policy improvement and an upper bound for the safety constraints for every agent. Furthermore, we propose a novel scalable and theoretically justified multi-agent constrained policy optimization method that follows a sequential update scheme to optimize $\kappa$-hop policies. Finally, we introduce a practical constrained policy optimization algorithm called Scal-MAPPO-L and experimentally validate the effectiveness of the proposed algorithm on a collection of benchmark tasks.

## Acknowledgements

This work is supported by the National Key Research and Development Program of China (No.2020AAA0106100), the National Natural Science Project of China (Nos.62276160, 62376013), and the Basic Research Program of Shanxi Province (No.202203021211294).

## Footnotes

\*Correspondence to `<weiwei@sxu.edu.cn>`.

## References

[1] Lukas Brunke, Melissa Greeff, Adam W Hall, Zhaocong Yuan, Siqi Zhou, Jacopo Panerati, and Angela P Schoellig. Safe learning in robotics: From learning-based control to safe reinforcement learning. *Annual Review of Control, Robotics, & Autonomous Systems*, 5:411–444, 2022.

[2] Shangding Gu, Long Yang, Yali Du, Guang Chen, Florian Walter, Jun Wang, Yaodong Yang, and Alois Knoll. A review of safe reinforcement learning: Methods, theory and applications. *arXiv preprint arXiv:2205.10330*, 2022.

[3] Wei Zhou, Dong Chen, Jun Yan, Zhaojian Li, Huilin Yin, and Wanchen Ge. Multi-agent reinforcement learning for cooperative lane changing of connected and autonomous vehicles in mixed traffic. *Autonomous Intelligent Systems*, 2(1):5, 2022.

[4] Wenqi Cui, Jiayi Li, and Baosen Zhang. Decentralized safe reinforcement learning for inverter-based voltage control. *Electric Power Systems Research*, 211:108609, 2022.

[5] Yu-Jia Chen, Deng-Kai Chang, and Cheng Zhang. Autonomous tracking using a swarm of uavs: A constrained multi-agent reinforcement learning approach. *IEEE Transactions on Vehicular Technology*, 69(11):13702–13717, 2020.

[6] Kai-Chieh Hsu, Allen Z Ren, Duy P Nguyen, Anirudha Majumdar, and Jaime F Fisac. Sim-to-lab-to-real: Safe reinforcement learning with shielding and generalization guarantees. *Artificial Intelligence*, 314:103811, 2023.

[7] Wenbo Zhang, Osbert Bastani, and Vijay Kumar. Mamps: Safe multi-agent reinforcement learning via model predictive shielding. *arXiv preprint arXiv:1910.12639*, 2019.

[8] Daniel Melcer, Christopher Amato, and Stavros Tripakis. Shield decentralization for safe multi-agent reinforcement learning. In *NeurIPS*, 2022.

[9] Chenyi Liu, Nan Geng, Vaneet Aggarwal, Tian Lan, Yuan Yang, and Mingwei Xu. Cmix: Deep multi-agent reinforcement learning with peak and average constraints. In *ECML-PKDD*, 2021.

[10] Shangding Gu, Jakub Grudzien Kuba, Yuanpei Chen, Yali Du, Long Yang, Alois Knoll, and Yaodong Yang. Safe multi-agent reinforcement learning for multi-robot control. *Artificial Intelligence*, 319:103905, 2023.

[11] Songtao Lu, Kaiqing Zhang, Tianyi Chen, Tamer Başar, and Lior Horesh. Decentralized policy gradient descent ascent for safe multi-agent reinforcement learning. In *AAAI*, 2021.

[12] Donghao Ying, Yunkai Zhang, Yuhao Ding, Alec Koppel, and Javad Lavaei. Scalable primal-dual actor-critic method for safe multi-agent rl with general utilities. In *NuerIPS*, 2023.

[13] Amir Dembo and Andrea Montanari. Gibbs measures and phase transitions on sparse random graphs. *arXiv preprint arXiv:0910.5460*, 2009.

[14] David Gamarnik. Correlation decay method for decision, optimization, and inference in large-scale networks. In *Theory Driven by Influential Applications*, pages 108–121. 2013.

[15] Guannan Qu, Yiheng Lin, Adam Wierman, and Na Li. Scalable multi-agent reinforcement learning for networked systems with average reward. In *NeurIPS*, 2020.

[16] Jakub Grudzien Kuba, Ruiqing Chen, Muning Wen, Ying Wen, Fanglei Sun, Jun Wang, and Yaodong Yang. Trust region policy optimisation in multi-agent reinforcement learning. In *ICLR*, 2022.

[17] John Schulman, Sergey Levine, Pieter Abbeel, Michael Jordan, and Philipp Moritz. Trust region policy optimization. In *ICML*, 2015.

[18] John Schulman, Filip Wolski, Prafulla Dhariwal, Alec Radford, and Oleg Klimov. Proximal policy optimization algorithms. *arXiv preprint arXiv:1707.06347*, 2017.

[19] Bei Peng, Tabish Rashid, Christian Schroeder de Witt, Pierre-Alexandre Kamienny, Philip Torr, Wendelin Boehmer, and Shimon Whiteson. Facmac: Factored multi-agent centralised policy gradients. In *NeurIPS*, 2021.

[20] Chao Yu, Akash Velu, Eugene Vinitsky, Jiaxuan Gao, Yu Wang, Alexandre Bayen, and Yi Wu. The surprising effectiveness of ppo in cooperative multi-agent games. In *NeurIPS*, 2022.

[21] Javier García and Diogo Shafie. Teaching a humanoid robot to walk faster through safe reinforcement learning. *Engineering Applications of Artificial Intelligence*, 88:103360, 2020.

[22] Shounak Datta, Yanjun Li, Matthew M Ruppert, Yuanfang Ren, Benjamin Shickel, Tezcan Ozrazgat-Baslanti, Parisa Rashidi, and Azra Bihorac. Reinforcement learning in surgery. *Surgery*, 170(1):329–332, 2021.

[23] Shangding Gu, Guang Chen, Lijun Zhang, Jing Hou, Yingbai Hu, and Alois Knoll. Constrained reinforcement learning for vehicle motion planning with topological reachability analysis. *Robotics*, 11(4):81, 2022.

[24] Dongsheng Ding, Kaiqing Zhang, Tamer Basar, and Mihailo R Jovanovic. Natural policy gradient primal-dual method for constrained markov decision processes. In *NeurIPS*, 2020.

[25] Osbert Bastani, Yewen Pu, and Armando Solar-Lezama. Verifiable reinforcement learning via policy extraction. In *NeurIPS*, 2018.

[26] Yinlam Chow, Ofir Nachum, Edgar Duenez-Guzman, and Mohammad Ghavamzadeh. A lyapunov-based approach to safe reinforcement learning. In *NeurIPS*, 2018.

[27] Yanan Sui, Alkis Gotovos, Joel Burdick, and Andreas Krause. Safe exploration for optimization with gaussian processes. In *ICML*, 2015.

[28] Aivar Sootla, Alexander I Cowen-Rivers, Taher Jafferjee, Ziyan Wang, David H Mguni, Jun Wang, and Haitham Ammar. Sauté rl: Almost surely safe reinforcement learning using state augmentation. In *ICML*, 2022.

[29] Peter Sunehag, Guy Lever, Audrunas Gruslys, Wojciech Marian Czarnecki, Vinicius Zambaldi, Max Jaderberg, Marc Lanctot, Nicolas Sonnerat, Joel Z Leibo, Karl Tuyls, et al. Value-decomposition networks for cooperative multi-agent learning. *arXiv preprint arXiv:1706.05296*, 2017.

[30] Tabish Rashid, Mikayel Samvelyan, Christian Schroeder, Gregory Farquhar, Jakob Foerster, and Shimon Whiteson. Qmix: Monotonic value function factorisation for deep multi-agent reinforcement learning. In *ICML*, 2018.

[31] Sven Gronauer and Klaus Diepold. Multi-agent deep reinforcement learning: a survey. *Artificial Intelligence Review*, pages 1–49, 2022.

[32] Volodymyr Mnih, Adria Puigdomenech Badia, Mehdi Mirza, Alex Graves, Timothy Lillicrap, Tim Harley, David Silver, and Koray Kavukcuoglu. Asynchronous methods for deep reinforcement learning. In *ICML*, 2016.

[33] S Whiteson, M Samvelyan, T Rashid, CS De Witt, G Farquhar, N Nardelli, TGJ Rudner, CM Hung, PHS Torr, and J Foerster. The starcraft multi-agent challenge. In *AAMAS*, 2019.

[34] Wei Du and Shifei Ding. A survey on multi-agent deep reinforcement learning: from the perspective of challenges and applications. *Artificial Intelligence Review*, 54:3215–3238, 2021.

[35] Afshin Oroojlooy and Davood Hajinezhad. A review of cooperative multi-agent deep reinforcement learning. *Applied Intelligence*, 53(11):13677–13722, 2023.

[36] Kaiqing Zhang, Zhuoran Yang, and Tamer Başar. Multi-agent reinforcement learning: A selective overview of theories and algorithms. *Studies in Systems, Decision and Control*, pages 321–384, 2021.

[37] Ming Tan. Multi-agent reinforcement learning: Independent vs. cooperative agents. In *ICML*, 1993.

[38] Ardi Tampuu, Tambet Matiisen, Dorian Kodelja, Ilya Kuzovkin, Kristjan Korjus, Juhan Aru, Jaan Aru, and Raul Vicente. Multiagent cooperation and competition with deep reinforcement learning. *arXiv preprint arXiv:1511.08779*, 2015.

[39] Christian Schroeder de Witt, Tarun Gupta, Denys Makoviichuk, Viktor Makoviychuk, Philip HS Torr, Mingfei Sun, and Shimon Whiteson. Is independent learning all you need in the starcraft multi-agent challenge? *arXiv preprint arXiv:2011.09533*, 2020.

[40] Jiechuan Jiang, Chen Dun, Tiejun Huang, and Zongqing Lu. Graph convolutional reinforcement learning. In *ICLR*, 2019.

[41] Tianshu Chu, Sandeep Chinchali, and Sachin Katti. Multi-agent reinforcement learning for networked system control. In *ICLR*, 2019.

[42] Guannan Qu, Adam Wierman, and Na Li. Scalable reinforcement learning for multi-agent networked systems. *arXiv preprint arXiv:1912.02906*, 2019.

[43] Yiheng Lin, Guannan Qu, Longbo Huang, and Adam Wierman. Multi-agent reinforcement learning in stochastic networked systems. In *NeurIPS*, 2021.

[44] Hans-Otto Georgii. *Gibbs measures and phase transitions*. Walter de Gruyter GmbH & Co. KG, Berlin, 2011.

[45] Alison L Gibbs and Francis Edward Su. On choosing and bounding probability metrics. *International statistical review*, 70(3):419–435, 2002.

[46] Wenjun Mei, Shadi Mohagheghi, Sandro Zampieri, and Francesco Bullo. On the dynamics of deterministic epidemic propagation over networks. *Annual Reviews in Control*, 44:116–128, 2017.

[47] Alessandro Zocca. Temporal starvation in multi-channel csma networks: an analytical framework. *ACM SIGMETRICS Performance Evaluation Review*, 46(3):52–53, 2019.

[48] Guannan Qu and Na Li. Exploiting fast decaying and locality in multi-agent mdp with tree dependence structure. In *CDC*, 2019.

[49] Haotian Gu, Xin Guo, Xiaoli Wei, and Renyuan Xu. Mean-field controls with q-learning for cooperative marl: convergence and complexity analysis. *SIAM Journal on Mathematics of Data Science*, 3(4):1168–1196, 2021.

[50] Yaodong Yang, Rui Luo, Minne Li, Ming Zhou, Weinan Zhang, and Jun Wang. Mean field multi-agent reinforcement learning. In *NeurIPS*, 2018.

[51] Changxi Zhu, Mehdi Dastani, and Shihan Wang. A survey of multi-agent reinforcement learning with communication. *arXiv preprint arXiv:2203.08975*, 2022.

[52] Junjie Sheng, Xiangfeng Wang, Bo Jin, Junchi Yan, Wenhao Li, Tsung-Hui Chang, Jun Wang, and Hongyuan Zha. Learning structured communication for multi-agent reinforcement learning. *Autonomous Agents and Multi-Agent Systems*, 36(2):50, 2022.

# A   Preliminary lemmas

Before proving propositions, corollaries, and theorems, we need a series of intermediate results as a foundation. Results similar to Lemmas A.1 and A.2 can be found in Chapter 8 of [44], Lemma A.3 is an extension of results from [15], Lemma A.4 is an extension of Lemma A.2 found in [12]. We state these lemmas and provide the corresponding proofs for completeness as follows.

**Lemma A.1.** *Let* $f : \mathcal{S} \rightarrow [m, M]$*, where* $\mathcal{S} = \times_{i \in \mathcal{N}} \mathcal{S}^i$ *and* $m, M \in \mathbb{R}$*. For every* $i \in \mathcal{N}$*, let* $\mu^i$ *and* $\nu^i$ *be two distributions on* $\mathcal{S}^i$*. Let* $\boldsymbol{\mu}$ *and* $\boldsymbol{\nu}$ *be the respective product distributions. Let* $\delta^i(f(\mathbf{s})) = \sup_{\mathbf{s}^i, \mathbf{s}^{-i}, \mathbf{s}'^i} \left| f\left(\mathbf{s}^i, \mathbf{s}^{-i}\right) - f\left(\mathbf{s}'^i, \mathbf{s}^{-i}\right) \right|$*. Then, one have*

$$\left| \mathbb{E}_{\mathbf{s} \sim \boldsymbol{\mu}} f(\mathbf{s}) - \mathbb{E}_{\mathbf{s} \sim \boldsymbol{\nu}} f(\mathbf{s}) \right| \le \sum_{i \in \mathcal{N}} D_{\mathrm{TV}} \left( \mu^i, \nu^i \right) \delta^i(f). \tag{21}$$

*Proof.* We prove Lemma A.1 by induction. Note that

$$D_{\mathrm{TV}}(\boldsymbol{\mu}, \boldsymbol{\nu}) = \frac{1}{2} \max_{|h| \le 1} |\mathbb{E}_{\boldsymbol{\mu}}(h) - \mathbb{E}_{\boldsymbol{\nu}}(h)|$$

is an equivalent formulation of the total variation distance [45].

For $|\mathcal{N}| = 1$, one have

$$\left| \mathbb{E}_{\mu^1}(f) - \mathbb{E}_{\nu^1}(f) \right|$$

$$= \left| \mathbb{E}_{\mu^1} \left( f - \frac{M+m}{2} \right) - \mathbb{E}_{\nu^1} \left( f - \frac{M+m}{2} \right) \right|$$

$$= \frac{M-m}{2} \left| \mathbb{E}_{\mu^1} \left( \frac{2f}{M-m} - \frac{M+m}{M-m} \right) - \mathbb{E}_{\nu^1} \left( \frac{2f}{M-m} - \frac{M+m}{M-m} \right) \right|$$

$$\le \frac{M-m}{2} \max_{|h| \le 1} \left| \mathbb{E}_{\mu^1}(h) - \mathbb{E}_{\nu^1}(h) \right|$$

$$= D_{\mathrm{TV}} \left( \mu^1, \nu^1 \right) \delta^1(f).$$

As induction assumption, assume that Lemma A.1 holds for $|\mathcal{N}| > 1$. Then, one have

$$\left| \mathbb{E}_{\mathbf{s} \sim \boldsymbol{\mu}} f(\mathbf{s}) - \mathbb{E}_{\mathbf{s} \sim \boldsymbol{\nu}} f(\mathbf{s}) \right|$$

$$= \left| \mathbb{E}_{\mathrm{s}^1 \sim \mu^1} \mathbb{E}_{\mathbf{s}^{2:n} \sim \boldsymbol{\mu}^{2:n}} f(\mathbf{s}) - \mathbb{E}_{\mathrm{s}^1 \sim \nu^1} \mathbb{E}_{\mathbf{s}^{2:n} \sim \boldsymbol{\nu}^{2:n}} f(\mathbf{s}) \right|$$

$$= \left| \mathbb{E}_{\mathrm{s}^1 \sim \mu^1} \mathbb{E}_{\mathbf{s}^{2:n} \sim \boldsymbol{\mu}^{2:n}} f(\mathbf{s}) - \mathbb{E}_{\mathrm{s}^1 \sim \mu^1} \mathbb{E}_{\mathbf{s}^{2:n} \sim \boldsymbol{\nu}^{2:n}} f(\mathbf{s}) \right.$$

$$\left. + \mathbb{E}_{\mathrm{s}^1 \sim \mu^1} \mathbb{E}_{\mathbf{s}^{2:n} \sim \boldsymbol{\nu}^{2:n}} f(\mathbf{s}) - \mathbb{E}_{\mathrm{s}^1 \sim \nu^1} \mathbb{E}_{\mathbf{s}^{2:n} \sim \boldsymbol{\nu}^{2:n}} f(\mathbf{s}) \right|$$

$$\le \left| \mathbb{E}_{\mathrm{s}^1 \sim \mu^1} \mathbb{E}_{\mathbf{s}^{2:n} \sim \boldsymbol{\mu}^{2:n}} f(\mathbf{s}) - \mathbb{E}_{\mathrm{s}^1 \sim \mu^1} \mathbb{E}_{\mathbf{s}^{2:n} \sim \boldsymbol{\nu}^{2:n}} f(\mathbf{s}) \right|$$

$$+ \left| \mathbb{E}_{\mathrm{s}^1 \sim \mu^1} \mathbb{E}_{\mathbf{s}^{2:n} \sim \boldsymbol{\nu}^{2:n}} f(\mathbf{s}) - \mathbb{E}_{\mathrm{s}^1 \sim \nu^1} \mathbb{E}_{\mathbf{s}^{2:n} \sim \boldsymbol{\nu}^{2:n}} f(\mathbf{s}) \right|$$

$$\le \mathbb{E}_{\mathrm{s}^1 \sim \mu^1} \left| \mathbb{E}_{\mathbf{s}^{2:n} \sim \boldsymbol{\mu}^{2:n}} f(\mathbf{s}) - \mathbb{E}_{\mathbf{s}^{2:n} \sim \boldsymbol{\nu}^{2:n}} f(\mathbf{s}) \right| + \left| \mathbb{E}_{\mathrm{s}^1 \sim \mu^1} \widetilde{f}\left(\mathrm{s}^1\right) - \mathbb{E}_{\mathrm{s}^1 \sim \nu^1} \widetilde{f}\left(\mathrm{s}^1\right) \right|,$$

where $\widetilde{f}\left(\mathrm{s}^1\right) = \mathbb{E}_{\mathbf{s}^{2:n} \sim \boldsymbol{\nu}^{2:n}} f(\mathbf{s})$.

By induction assumption, one have

$$\left| \mathbb{E}_{\mathbf{s}^{2:n} \sim \boldsymbol{\mu}^{2:n}} f(\mathbf{s}) - \mathbb{E}_{\mathbf{s}^{2:n} \sim \boldsymbol{\nu}^{2:n}} f(\mathbf{s}) \right| \le \sum_{i \ne 1 \in \mathcal{N}} D_{\mathrm{TV}} \left( \mu^1, \nu^1 \right) \delta^1 \left( f\left(\mathrm{s}^1, \cdot\right) \right)$$

$$\le \sum_{i \ne 1 \in \mathcal{N}} D_{\mathrm{TV}} \left( \mu^1, \nu^1 \right) \delta^1(f).$$

Since

$$\delta^1(\widetilde{f}) = \sup_{\mathrm{s}^1, \mathrm{s}'^1} \left| \mathbb{E}_{\mathbf{s}^{2:n} \sim \boldsymbol{\nu}^{2:n}} f\left(\mathrm{s}^1, \mathbf{s}^{2:n}\right) - \mathbb{E}_{\mathbf{s}^{2:n} \sim \boldsymbol{\nu}^{2:n}} f\left(\mathrm{s}'^1, \mathbf{s}^{2:n}\right) \right|$$

$$\le \sup_{\mathrm{s}^1, \mathrm{s}'^1} \mathbb{E}_{\mathbf{s}^{2:n} \sim \boldsymbol{\nu}^{2:n}} \left| f\left(\mathrm{s}^1, \mathbf{s}^{2:n}\right) - f\left(\mathrm{s}'^1, \mathbf{s}^{2:n}\right) \right|$$

$$\le \sup_{\mathrm{s}^1, \mathrm{s}'^1, \mathbf{s}^{2:n}} \left| f\left(\mathrm{s}^1, \mathbf{s}^{2:n}\right) - f\left(\mathrm{s}'^1, \mathbf{s}^{2:n}\right) \right|$$

$$= \delta^1(f),$$

one have

$$|\mathbb{E}_{\mathbf{s}\sim\boldsymbol{\mu}}f(\mathbf{s}) - \mathbb{E}_{\mathbf{s}\sim\boldsymbol{\nu}}f(\mathbf{s})|$$

$$\leq \mathbb{E}_{\mathrm{s}^1\sim\mu^1} \sum_{i\neq 1\in\mathcal{N}} D_{\mathrm{TV}}\left(\mu^i,\nu^i\right)\delta^i(f) + D_{\mathrm{TV}}\left(\mu^i,\nu^i\right)\delta^i(f)$$

$$\leq \sum_{i\in\mathcal{N}} D_{\mathrm{TV}}\left(\mu^i,\nu^i\right)\delta^i(f),$$

which concludes the induction. $\qquad\square$

**Lemma A.2.** *Consider a Markov Chain with state $\mathbf{s}\in\mathcal{S}$, where $\mathcal{S}=\times_{i\in\mathcal{N}}\mathcal{S}^i$, and $\mathcal{N}=\{1,\ldots,n\}$ is the set of agents. Suppose its transition probability factorizes as*

$$P(\mathbf{s}_{t+1}\mid\mathbf{s}_t) = \prod_{i\in\mathcal{N}} P^i\left(\mathbf{s}_{t+1}^i\mid\mathbf{s}_t\right).$$

*Let $W\in\mathbb{R}^{n\times n}$ be a matrix whose elements respect the condition*

$$W^{ij} \geq \sup_{\mathrm{s}^j,\mathbf{s}^{-j},\mathrm{s}'^j} D_{\mathrm{TV}}\left(P^i\left(\cdot\mid \mathrm{s}^j,\mathbf{s}^{-j}\right), P^i\left(\cdot\mid \mathrm{s}'^j,\mathbf{s}^{-j}\right)\right).$$

*If $\sum_{j\in\mathcal{J}} e^{\beta d(j,i)}W^{ij}\leq\zeta$, $\mathcal{J}\subseteq\mathcal{N}$, then one have*

$$\sup_{\mathrm{s}^j,\mathbf{s}^{-j},\mathrm{s}'^j} D_{\mathrm{TV}}\left(P^i\left(\cdot\mid \mathbf{s}^J,\mathbf{s}^{-J}\right), P^i\left(\cdot\mid \mathbf{s}'^J,\mathbf{s}^{-J}\right)\right) \leq \sum_{j\in\mathcal{J}} W^{ij}, \qquad (22)$$

*and*

$$\sup_{\mathrm{s}^j,\mathbf{s}^{-j},\mathrm{s}'^j} D_{\mathrm{TV}}\left(P^i\left(\cdot\mid \mathbf{s}^J,\mathbf{s}^{-J}\right), P^i\left(\cdot\mid \mathbf{s}'^J,\mathbf{s}^{-J}\right)\right) \leq \zeta e^{-\beta d(\mathcal{J},i)}, \qquad (23)$$

*where $d(\mathcal{J},i) = \min_{j\in\mathcal{J}} d(j,i)$.*

*Proof.* We prove the first claim of Lemma A.2. The first claim clearly holds if $|\mathcal{J}|=1$. As induction assumption, assume that the first claim holds for a set $\mathcal{J}$. Then, it holds for $\mathcal{J}'=\mathcal{J}+\{k\}$

$$\sup_{\mathrm{s}^j,\mathbf{s}^{-j},\mathrm{s}'^j} D_{\mathrm{TV}}\left(P^i\left(\cdot\mid \mathbf{s}^{J'},\mathbf{s}^{-J'}\right), P^i\left(\cdot\mid \mathbf{s}'^{J'},\mathbf{s}^{-J'}\right)\right)$$

$$= \sup_{\substack{A\subseteq\mathcal{S}^i\\ \mathrm{s}^j,\mathbf{s}^{-j},\mathrm{s}'^j}} \left|P^i\left(A\mid \mathbf{s}^{J'},\mathbf{s}^{-J'}\right), P^i\left(A\mid \mathbf{s}'^{J'},\mathbf{s}^{-J'}\right)\right|$$

$$\leq \sup_{\substack{A\subseteq\mathcal{S}^i\\ \mathrm{s}^j,\mathbf{s}^{-j},\mathrm{s}'^j}} \left|P^i\left(A\mid \mathbf{s}^{J'},\mathbf{s}^{-J'}\right), P^i\left(A\mid \mathbf{s}'^{J},\mathbf{s}^{-J}\right)\right|$$

$$+ \sup_{\substack{A\subseteq\mathcal{S}^i\\ \mathrm{s}^j,\mathbf{s}^{-j},\mathrm{s}'^j}} \left|P^i\left(A\mid \mathbf{s}'^{J},\mathbf{s}^{-J}\right), P^i\left(A\mid \mathbf{s}'^{J'},\mathbf{s}^{-J'}\right)\right|$$

$$\leq \sum_{j\in\mathcal{J}} W^{ij} + W^{ik}$$

$$= \sum_{j\in\mathcal{J}'} W^{ij}.$$

The second claim follows immediately, since

$$e^{\beta d(\mathcal{J},i)}\sum_{j\in\mathcal{J}} W^{ij} \leq \sum_{j\in\mathcal{J}} e^{\beta d(j,i)}W^{ij} \leq \sum_{j\in\mathcal{N}} e^{\beta d(j,i)}W^{ij} \leq \zeta,$$

and

$$\sum_{j\in\mathcal{J}} W^{ij} \leq \zeta e^{-\beta d(\mathcal{J},i)}.$$

$\qquad\square$

**Lemma A.3.** *Consider the setting of Lemma A.2. For a generic value of $\kappa$, denote by $\rho_t$ and $\widetilde{\rho}_t$ the distribution of $\mathbf{s}_t$ with starting state, respectively, $\mathbf{s} = \left(s_{\mathcal{N}_\kappa^i}, \mathbf{s}_{\mathcal{N}_\kappa^{-i}}\right)$ and $\widetilde{\mathbf{s}} = \left(s_{\mathcal{N}_\kappa^i}, \widetilde{\mathbf{s}}_{\mathcal{N}_\kappa^{-i}}\right)$. Then, if $\sum_{j\in\mathcal{N}} e^{\beta d(j,i)} W^{ij} \leq \zeta$, we have that $D_{\mathrm{TV}}\left(\rho_t^i, \widetilde{\rho}_t^i\right) \leq \zeta^t e^{-\beta\kappa}, \forall\, i \in \mathcal{N}$.*

*Proof.* We prove Lemma A.3 by induction. The case where $t = 1$ follows from Lemma A.2. As induction assumption, assume that Lemma A.3 holds for $t$. Then, one have

$$\left|\mathbb{E}_{\mathbf{s}\sim\boldsymbol{\rho}_{t+1}}\mathbf{1}_A(\mathbf{s}) - \mathbb{E}_{\mathbf{s}\sim\widetilde{\boldsymbol{\rho}}_{t+1}}\mathbf{1}_A(\mathbf{s})\right|$$

$$= \left|\mathbb{E}_{\mathbf{s}\sim\boldsymbol{\rho}_t}\mathbb{E}_{\mathbf{s}\sim P^i(\cdot|\mathbf{s})}\mathbf{1}_A(\mathbf{s}) - \mathbb{E}_{\mathbf{s}\sim\widetilde{\boldsymbol{\rho}}_t}\mathbb{E}_{\mathbf{s}\sim P^i(\cdot|\mathbf{s})}\mathbf{1}_A(\mathbf{s})\right|$$

$$\leq \sum_{j\in\mathcal{N}} D_{\mathrm{TV}}\left(\rho_t^i, \widetilde{\rho}_t^i\right) \delta^j \left(E_{\mathbf{s}\sim P^i(\cdot|\cdot)}\mathbf{1}_A(\mathbf{s})\right)$$

$$\leq \sum_{j\in\mathcal{N}} D_{\mathrm{TV}}\left(\rho_t^i, \widetilde{\rho}_t^i\right) W^{ij}$$

$$= \zeta^t e^{-\beta\kappa} \sum_{j\in\mathcal{N}} e^{\beta d(j,i)} W^{ij}$$

$$\leq \zeta^{t+1} e^{-\beta\kappa},$$

where we used Lemma A.1 in the first inequality. $\qquad\square$

**Lemma A.4.** *Consider the setting of Lemma A.2. Let $P^t\left(\mathbf{s}' \mid \mathbf{s}\right) = P\left(\mathbf{s}_t = \mathbf{s}' \mid \mathbf{s}_0 = \mathbf{s}\right)$ and*

$$\delta^j P^{i,t} = \sup_{\mathrm{s}^j, \mathbf{s}^{-j}, \mathrm{s}'^j} D_{\mathrm{TV}}\left(P^{i,t}\left(\cdot \mid \mathrm{s}^j, \mathbf{s}^{-j}\right), P^{i,t}\left(\cdot \mid \mathrm{s}'^j, \mathbf{s}^{-j}\right)\right).$$

*If $\sum_{j\in\mathcal{N}} e^{\beta d(i,j)} W^{ij} \leq \zeta$, we have*

$$\sum_{j\in\mathcal{N}} e^{\beta d(i,j)} \delta^j P^{i,t} \leq \zeta^t, \forall\, i \in \mathcal{N}. \tag{24}$$

*Proof.* We prove Lemma A.4 by induction. The claim holds for $t = 1$,

$$\sum_{j\in\mathcal{N}} e^{\beta d(i,j)} \delta^j P^{i,t} = \sum_{j\in\mathcal{N}} e^{\beta d(i,j)} W^{ij} \leq \zeta.$$

As induction assumption, we assume that the claim holds for $t$. Then, using Lemma A.1,

$$\delta^j P^{i,t+1} = \sup_{\substack{A\subseteq\mathcal{S}^i \\ \mathrm{s}^j, \mathbf{s}^{-j}, \mathrm{s}'^j}} \left|\mathbb{E}_{\mathbf{s}\sim P^{i,t+1}(\cdot|\mathrm{s}^j, \mathbf{s}^{-j})}\mathbf{1}_A(\mathbf{s}) - \mathbb{E}_{\mathbf{s}\sim P^{i,t+1}(\cdot|\mathrm{s}'^j, \mathbf{s}^{-j})}\mathbf{1}_A(\mathbf{s})\right|$$

$$= \sup_{\substack{A\subseteq\mathcal{S}'^i \\ \mathrm{s}^j, \mathbf{s}^{-j}, \mathrm{s}'^j}} \left|\mathbb{E}_{\mathbf{s}\sim P^t(\cdot|\mathrm{s}^j, \mathbf{s}^{-j})}\mathbb{E}_{\mathbf{s}\sim P^i(\cdot|\mathbf{s})}\mathbf{1}_A(\mathbf{s}) - \mathbb{E}_{\mathbf{s}\sim P^t(\cdot|\mathrm{s}'^j, \mathbf{s}^{-j})}\mathbb{E}_{\mathbf{s}\sim P^i(\cdot|\mathbf{s})}\mathbf{1}_A(\mathbf{s})\right|$$

$$\leq \sup_{\mathrm{s}^j, \mathbf{s}^{-j}, \mathrm{s}'^j} \sum_{k\in\mathcal{N}} D_{\mathrm{TV}}\left(P^{k,t}\left(\cdot \mid \mathrm{s}^j, \mathbf{s}^{-j}\right), P^{k,t}\left(\cdot \mid \mathrm{s}'^j, \mathbf{s}^{-j}\right)\right) \delta^j\left(E_{\mathbf{s}\sim P^i(\cdot|\cdot)}\mathbf{1}_A(\mathbf{s})\right)$$

$$\leq \sum_{k\in\mathcal{N}} \delta^j P^{k,t} W^{ik},$$

and using the inverse triangle inequality,

$$\sum_{j\in\mathcal{N}} e^{\beta d(i,j)} \delta^j P^{i,t+1} \leq \sum_{j\in\mathcal{N}} e^{\beta d(i,j)} \sum_{k\in\mathcal{N}} \delta^j P^{k,t} W^{ik}$$

$$\leq \sum_{k\in\mathcal{N}} e^{\beta d(i,k)} W^{ik} \sum_{j\in\mathcal{N}} e^{\beta(d(i,j)-d(i,k))} \delta^j P^{k,t}$$

$$\leq \sum_{k\in\mathcal{N}} e^{\beta d(i,k)} W^{ik} \sum_{j\in\mathcal{N}} e^{\beta d(k,j)} \delta^j P^{k,t}$$

$$\leq \zeta^{t+1},$$

which concludes the induction. $\qquad\square$

# B Supplementary materials for Section 2

## B.1 Spatial correlation decay

Exponential decay property [13, 14], also known as spatial correlation decay, is a powerful property associated with local interactions, which says that the impact of agents on each other decays exponentially in their graph distance. Over the past decades, many researchers have utilized spatial correlation property to design scalable, distributed algorithms for optimization and control problems in scenarios such as epidemics [46] and wireless communication [47]. Inspired by the studies mentioned above, a recent line of work [48] has formally considered spatial decay of correlation assumptions and proposes a method that finds nearly optimal local policies. An application [49] with the same principles adopts the setting of mean-field MARL [50], which proposes an actor-critic algorithm with global convergence. However, unlike the mean-field setting, which requires an agent's transition scheme to be only affected by the mean effect from its neighbors and effective only when agents are homogeneous, we allow each agent to have different transition probabilities and local policies.

## B.2 Regarding Assumptions 2.1 - 2.2

Assumption 2.1 portrays a common phenomenon: the transition dynamic of each agent is exponentially less sensitive to perturbations of the states and actions of more distant agents. This is commonly seen in scenarios involving wireless communication, epidemics, traffic, and so on [46, 47]. Assumption 2.2 imposes a design constraint for the policy class that encodes a weaker correlation decay property than the assumptions on the nature of Assumption 2.1. Moreover, Assumption 2.2 reveals how much information is lost compared with access to the global state and allows us to consider a policy class with the necessary properties for the optimal policy under Assumption 2.1. Below, we use a mathematical example to illustrate the relationship between the two assumptions.

**Mathematical example:** Firstly, we start from Assumption 2.1, letting $\widetilde{\kappa} = \max_{i,j \in \mathcal{N}} d(i,j)$ be the maximum distance between agent $i$ and agent $j$. Define a set of differentiable functions $\left\{ f_\kappa : \mathcal{S}_{\mathcal{N}_\kappa^i} \times \mathcal{A}^i \to \mathcal{K} \mid 0 \leq \kappa \leq \widetilde{\kappa} \right\}$, where $\mathcal{K} \subset [-K, K]$, $K > 0$, and a set of parameters $\{ \alpha_\kappa \geq 0 \mid 0 \leq \kappa \leq \widetilde{\kappa} \}$. Then, for each agent $i$, one have

$$f^i\left(\mathbf{s}, \mathrm{a}^i\right) = \sum_{\kappa=0}^{\widetilde{\kappa}} \alpha_\kappa f_\kappa^i\left(\mathrm{s}_{\mathcal{N}_\kappa^i}, \mathrm{a}^i\right),$$

$$\pi^i\left(\mathrm{a} \mid \mathbf{s}\right) = \frac{\exp\left(f^i\left(\mathbf{s}, \mathrm{a}\right)\right)}{\sum_{\mathrm{a}' \in \mathcal{A}^i} \exp\left(f^i\left(\mathbf{s}, \mathrm{a}'\right)\right)}.$$

By tuning the parameters $\alpha_\kappa$, we can make any policy belonging to this policy class respect Assumptions 2.2, as we show in the following. Let $\kappa \in \{0, \dots, \widetilde{\kappa}\}$, $\mathbf{s}, \widetilde{\mathbf{s}} \in \mathcal{S}$ be such that $\mathbf{s}_{\mathcal{N}_\kappa^i} = \widetilde{\mathbf{s}}_{\mathcal{N}_\kappa^i}$, then one have

$$\left\| \pi^i(\cdot|\mathbf{s}) - \pi^i(\cdot|\widetilde{\mathbf{s}}) \right\|_1$$

$$= \sum_{\mathrm{a} \in \mathcal{A}^i} \left| \pi^i(\mathrm{a} \mid \mathbf{s}) - \pi^i(\mathrm{a} \mid \widetilde{\mathbf{s}}) \right|$$

$$= \sum_{\mathrm{a} \in \mathcal{A}^i} \left| \frac{\exp\left(f^i\left(\mathbf{s}, \mathrm{a}\right)\right)}{\sum_{\mathrm{a}' \in \mathcal{A}^i} \exp\left(f^i\left(\mathbf{s}, \mathrm{a}'\right)\right)} - \frac{\exp\left(f^i\left(\widetilde{\mathbf{s}}, \mathrm{a}\right)\right)}{\sum_{\mathrm{a}' \in \mathcal{A}^i} \exp\left(f^i\left(\widetilde{\mathbf{s}}, \mathrm{a}'\right)\right)} \right|$$

$$= \frac{\sum_{\mathrm{a} \in \mathcal{A}^i} \left| \sum_{\mathrm{a}' \in \mathcal{A}^i} \exp\left(f^i\left(\mathbf{s}, \mathrm{a}\right)\right) \exp\left(f^i\left(\widetilde{\mathbf{s}}, \mathrm{a}'\right)\right) - \sum_{\mathrm{a}' \in \mathcal{A}^i} \exp\left(f^i\left(\widetilde{\mathbf{s}}, \mathrm{a}\right)\right) \exp\left(f^i\left(\mathbf{s}, \mathrm{a}'\right)\right) \right|}{\sum_{\mathrm{a}' \in \mathcal{A}^i} \exp\left(f^i\left(\mathbf{s}, \mathrm{a}'\right)\right) \sum_{\mathrm{a}' \in \mathcal{A}^i} \exp\left(f^i\left(\widetilde{\mathbf{s}}, \mathrm{a}'\right)\right)}$$

$$\leq \frac{\sum_{\mathrm{a} \in \mathcal{A}^i} \sum_{\mathrm{a}' \in \mathcal{A}^i} \left| \exp\left(f^i\left(\mathbf{s}, \mathrm{a}\right)\right) \exp\left(f^i\left(\widetilde{\mathbf{s}}, \mathrm{a}'\right)\right) - \exp\left(f^i\left(\widetilde{\mathbf{s}}, \mathrm{a}\right)\right) \exp\left(f^i\left(\mathbf{s}, \mathrm{a}'\right)\right) \right|}{\sum_{\mathrm{a}' \in \mathcal{A}^i} \exp\left(f^i\left(\mathbf{s}, \mathrm{a}'\right)\right) \sum_{\mathrm{a}' \in \mathcal{A}^i} \exp\left(f^i\left(\widetilde{\mathbf{s}}, \mathrm{a}'\right)\right)}$$

$$\leq \frac{\sum_{\mathrm{a} \in \mathcal{A}^i} \left| \exp\left(f^i(\widetilde{\mathbf{s}}, \mathrm{a})\right) - \exp\left(f^i(\mathbf{s}, \mathrm{a})\right) \right|}{\sum_{\mathrm{a} \in \mathcal{A}^i} \exp\left(f^i(\widetilde{\mathbf{s}}, \mathrm{a})\right)}$$

$$\leq \frac{\sum_{\mathrm{a} \in \mathcal{A}^i} \left| f^i(\widetilde{\mathbf{s}}, \mathrm{a}) - f^i(\mathbf{s}, \mathrm{a}) \right| \exp\left(\sup_{\mathbf{s}' \in \{\mathbf{s}, \widetilde{\mathbf{s}}\}} f^i\left(\mathbf{s}', \mathrm{a}\right)\right)}{\sum_{\mathrm{a} \in \mathcal{A}^i} \exp\left(f^i(\widetilde{\mathbf{s}}, \mathrm{a})\right)}$$

$$\leq e^{2K(\widetilde{\kappa}-\kappa)} \frac{\sum_{a\in\mathcal{A}^i}\left|f^i(\widetilde{\mathbf{s}},a)-f^i(\mathbf{s},a)\right|\exp\left(f^i(\widetilde{\mathbf{s}},a)\right)}{\sum_{a\in\mathcal{A}^i}\exp\left(f^i(\widetilde{\mathbf{s}},a)\right)}$$

$$\leq e^{2K(\widetilde{\kappa}-\kappa)}\mathbb{E}_{\pi^i}\left|\sum_{\kappa'=\kappa+1}^{\widetilde{\kappa}}\alpha_{\kappa'}\left(f^i_{\kappa'}\left(\widetilde{\mathbf{s}}_{\mathcal{N}^i_{\kappa'}},a\right)-f^i_{\kappa'}\left(\mathbf{s}_{\mathcal{N}^i_{\kappa'}},a\right)\right)\right|$$

$$\leq e^{2K(\widetilde{\kappa}-\kappa)}\sum_{\kappa'=\kappa+1}^{\widetilde{\kappa}}\alpha_{\kappa'}\mathbb{E}_{\pi^i_{\kappa}}\left|\left(f^i_{\kappa'}\left(\widetilde{\mathbf{s}}_{\mathcal{N}^i_{\kappa'}},a\right)-f^i_{\kappa'}\left(\mathbf{s}_{\mathcal{N}^i_{\kappa'}},a\right)\right)\right|$$

$$\leq 2Ke^{2K(\widetilde{\kappa}-\kappa)}\sum_{\kappa'=\kappa+1}^{\widetilde{\kappa}}\alpha_{\kappa'}.$$

Denote that $(\xi,\beta)=\left(2Ke^{2K\widetilde{\kappa}}\sum_{\kappa'=\kappa+1}^{\widetilde{\kappa}}\alpha_{\kappa'},2K\right)$, and setting the parameters $\{\alpha_{\kappa'}\}_{\kappa'\in\{\kappa+1,...,\widetilde{\kappa}\}}$ small enough ensures that the policy respects Assumption 2.2.

*Remark* B.1. The mathematical example illustrates the relationship between the Assumptions 2.1 and 2.2. It is evident from this mathematical example that Assumption 2.2 necessarily holds when Assumption 2.1 holds and the parameters $\xi$ and $\beta$ satisfy certain conditions. However, for the sake of more concise presentation, we treat it as a separate assumption.

*Remark* B.2. When Assumption 2.1 holds, the numerical example can provide a reference basis for selecting the values of the parameters in Assumption 2.2. However, accurately determining the spatial decay of correlation for the dynamics remains a challenging engineering task. In this paper, we empirically adopt conservative values.

*Remark* B.3. Assumption 2.2 implies that multi-agent environments must satisfy the requirement that the impact from far-away agents' states and actions is almost negligible for the agent's decision; in other words, an action of an agent has an instantaneous effect on the system only locally. We believe that this formulation realistically describes most multi-agent interactions in the real-world. Take multi-vehicle transportation as an example. For a vehicle traveling on the road, a far-away vehicle taking different actions or being in a different state will affect itself shortly thereafter, but the impact on the current policy is minimal. More examples are seen in wireless communication, epidemics, traffic, and other scenarios [46, 47].

*Remark* B.4. It is worth noting that the assumption of spatial correlation decay is not in direct conflict with well-known phenomena, e.g., Butterfly Effect, since two seemingly unrelated things can also have a significant impact on each other, generally occurring in different time domains.

## C  Supplementary materials for Section 3

### C.1  Basic definitions

Regarding the state value function and the state-action value function, we give the following definitions.

*Definition* C.1. We define the state value function and the state-action value function in terms of reward as

$$V_{\boldsymbol{\pi}}(\mathbf{s}) \triangleq \mathbb{E}_{\mathbf{a}\sim\boldsymbol{\pi}}\left[Q_{\boldsymbol{\pi}}(\mathbf{s},\mathbf{a})\right], \tag{25}$$

$$Q_{\boldsymbol{\pi}}(\mathbf{s},\mathbf{a}) \triangleq \mathbb{E}_{\mathbf{s}_{1:\infty}\sim\boldsymbol{p},\mathbf{a}_{1:\infty}\sim\boldsymbol{\pi}}\left[\sum_{t=0}^{\infty}\gamma^t\boldsymbol{R}(\mathbf{s}_t,\mathbf{a}_t)|\mathbf{s}_0=\mathbf{s},\mathbf{a}_0=\mathbf{a}\right]. \tag{26}$$

Based on C.1, one can expand to derive

$$V^i_{\boldsymbol{\pi}}(\mathbf{s}) = \mathbb{E}_{\mathbf{a}^{-i}\sim\boldsymbol{\pi}^{-i}}\left[Q^i_{\boldsymbol{\pi}}(\mathbf{s},a^i)\right], \tag{27}$$

$$Q^i_{\boldsymbol{\pi}}(\mathbf{s},a^i) = \mathbb{E}_{\mathbf{a}^{-i}\sim\boldsymbol{\pi}^{-i},\mathbf{s}_{1:\infty}\sim\boldsymbol{p},\mathbf{a}_{1:\infty}\sim\boldsymbol{\pi}}\left[\sum_{t=0}^{\infty}\gamma^t\boldsymbol{R}(\mathbf{s}_t,a^i_t)|\mathbf{s}_0=\mathbf{s},\mathbf{a}_0=\mathbf{a}\right]. \tag{28}$$

*Definition* C.2. We define the $j_{th}$ state cost value function and state-action cost value function for agent $i$ as follows

$$V_{j,\boldsymbol{\pi}}^i(\mathbf{s}) \triangleq \mathbb{E}_{\mathbf{s}_{1:\infty}\sim\boldsymbol{p},\mathbf{a}_{1:\infty}\sim\boldsymbol{\pi}}\left[\sum_{t=0}^{\infty}\gamma^t C_j^i(\mathbf{s}_t,\mathbf{a}_t^i)|\mathbf{s}_0=\mathbf{s}\right], \tag{29}$$

$$Q_{j,\boldsymbol{\pi}}^i(\mathbf{s},\mathbf{a}^i) \triangleq \mathbb{E}_{\mathbf{a}^{-i}\sim\boldsymbol{\pi}^{-i},\mathbf{s}_{1:\infty}\sim\boldsymbol{p},\mathbf{a}_{1:\infty}\sim\boldsymbol{\pi}}\left[\sum_{t=0}^{\infty}\gamma^t C_j^i(\mathbf{s}_t,\mathbf{a}_t^i)|\mathbf{s}_0=\mathbf{s},\mathbf{a}_0=\mathbf{a}\right]. \tag{30}$$

## C.2 The proof of Lamma 3.1

*Proof.* We write the multi-agent advantage function as in its definition, and then expand it in a telescoping sum.

$$\begin{aligned} A_{\boldsymbol{\pi}}(\mathbf{s},\mathbf{a}) &= Q_{\boldsymbol{\pi}}(\mathbf{s},\mathbf{a}) - V_{\boldsymbol{\pi}}(\mathbf{s}) \\ &= \sum_{i=1}^{n}\left[Q_{\boldsymbol{\pi}}^{1:i}(\mathbf{s},\mathbf{a}^{1:i}) - Q_{\boldsymbol{\pi}}^{1:i-1}(\mathbf{s},\mathbf{a}^{1:i-1})\right] \\ &= \sum_{i=1}^{n}A_{\boldsymbol{\pi}}^i(\mathbf{s},\mathbf{a}^{1:i-1},\mathbf{a}^i). \end{aligned}$$

$\square$

## C.3 The proof of Proposition 3.3

*Proof.* Let $\mathbf{s},\widetilde{\mathbf{s}} \in \mathcal{S}$, $\mathbf{a},\widetilde{\mathbf{a}} \in \mathcal{A}$, such that for any agent $i \in \mathcal{N}$, $\mathbf{s}_{\mathcal{N}_\kappa^i} = \widetilde{\mathbf{s}}_{\mathcal{N}_\kappa^i}$ and $\mathbf{a}_{\mathcal{N}_\kappa^i} = \widetilde{\mathbf{a}}_{\mathcal{N}_\kappa^i}$. According to Equation (7), when only the state and action of the far-away agent are different, one have

$$\begin{aligned} &\left|A_{\boldsymbol{\pi}}^i(\mathbf{s},\mathbf{a}) - A_{\boldsymbol{\pi}}^i(\widetilde{\mathbf{s}},\widetilde{\mathbf{a}})\right| \\ &= \left|(Q_{\boldsymbol{\pi}}^i(\mathbf{s},\mathbf{a}) - V_{\boldsymbol{\pi}}^i(\mathbf{s})) - (Q_{\boldsymbol{\pi}}^i(\widetilde{\mathbf{s}},\widetilde{\mathbf{a}}) - V_{\boldsymbol{\pi}}^i(\widetilde{\mathbf{s}}))\right| \\ &= \left|(Q_{\boldsymbol{\pi}}^i(\mathbf{s},\mathbf{a}) - Q_{\boldsymbol{\pi}}^i(\widetilde{\mathbf{s}},\widetilde{\mathbf{a}})) + (V_{\boldsymbol{\pi}}^i(\mathbf{s}) - V_{\boldsymbol{\pi}}^i(\widetilde{\mathbf{s}}))\right| \\ &\leq \left|Q_{\boldsymbol{\pi}}^i(\mathbf{s},\mathbf{a}) - Q_{\boldsymbol{\pi}}^i(\widetilde{\mathbf{s}},\widetilde{\mathbf{a}})\right| + \left|V_{\boldsymbol{\pi}}^i(\mathbf{s}) - V_{\boldsymbol{\pi}}^i(\widetilde{\mathbf{s}})\right|. \end{aligned} \tag{31}$$

Next, we analyze $\left|Q_{\boldsymbol{\pi}}^i(\mathbf{s},\mathbf{a}) - Q_{\boldsymbol{\pi}}^i(\widetilde{\mathbf{s}},\widetilde{\mathbf{a}})\right|$ and $\left|V_{\boldsymbol{\pi}}^i(\mathbf{s}) - V_{\boldsymbol{\pi}}^i(\widetilde{\mathbf{s}})\right|$ separately.

Firstly, for $\left|Q_{\boldsymbol{\pi}}^i(\mathbf{s},\mathbf{a}) - Q_{\boldsymbol{\pi}}^i(\widetilde{\mathbf{s}},\widetilde{\mathbf{a}})\right|$, we have

$$\begin{aligned} &\left|Q_{\boldsymbol{\pi}}^i(\mathbf{s},\mathbf{a}) - Q_{\boldsymbol{\pi}}^i(\widetilde{\mathbf{s}},\widetilde{\mathbf{a}})\right| \\ &= \left|\sum_{t=0}^{\infty}\gamma^t\mathbb{E}\left[\boldsymbol{R}(\mathbf{s}_t,\mathbf{a}_t)\mid\boldsymbol{\pi},\mathbf{s}_0=\mathbf{s},\mathbf{a}_0=\mathbf{a}\right] - \sum_{t=0}^{\infty}\gamma^t\mathbb{E}\left[\boldsymbol{R}(\mathbf{s}_t,\mathbf{a}_t)\mid\boldsymbol{\pi},\mathbf{s}_0=\widetilde{\mathbf{s}},\mathbf{a}_0=\widetilde{\mathbf{a}}\right]\right| \\ &\leq \sum_{t=0}^{\infty}\gamma^t\left|\mathbb{E}\left[\boldsymbol{R}(\mathbf{s}_t,\mathbf{a}_t)\mid\boldsymbol{\pi},\mathbf{s}_0=\mathbf{s},\mathbf{a}_0=\mathbf{a}\right] - \mathbb{E}\left[\boldsymbol{R}(\mathbf{s}_t,\mathbf{a}_t)\mid\boldsymbol{\pi},\mathbf{s}_0=\widetilde{\mathbf{s}},\mathbf{a}_0=\widetilde{\mathbf{a}}\right]\right| \\ &\leq \sum_{t=1}^{\infty}\gamma^t D_{\mathrm{TV}}\left(\rho_t^i,\widetilde{\rho}_t^i\right), \end{aligned}$$

where $\rho_t^i$ and $\widetilde{\rho}_t^i$ are the distributions at time $t$ with starting point $(\mathbf{s},\mathbf{a})$ and $(\widetilde{\mathbf{s}},\widetilde{\mathbf{a}})$, respectively. We use the result in Lemma A.3 to bound $D_{\mathrm{TV}}\left(\rho_t^i,\widetilde{\rho}_t^i\right)$. The structure of our MDP implies that:

$$P(\mathbf{s}_{t+1},\mathbf{a}_{t+1}\mid\mathbf{s}_t,\mathbf{a}_t) = \prod_{i\in\mathcal{N}}\pi^i\left(\mathbf{a}_{t+1}^i\mid\mathbf{s}_{\mathcal{N}_\kappa^i,t+1}\right)P^i\left(\mathbf{s}_{t+1}^i\mid\mathbf{s}_{\mathcal{N}_\kappa^i,t},\mathbf{a}_t^i\right).$$

Then, if Assumption 2.1 holds, the requirements of Lemma A.3 are satisfied, one have $D_{\mathrm{TV}}\left(\rho_t^i,\widetilde{\rho}_t^i\right) \leq \zeta^t e^{-\beta\kappa}$ and

$$\left|Q_{\boldsymbol{\pi}}^i(\mathbf{s},\mathbf{a}) - Q_{\boldsymbol{\pi}}^i(\widetilde{\mathbf{s}},\widetilde{\mathbf{a}})\right| \leq \sum_{t=1}^{\infty}\gamma^t D_{\mathrm{TV}}\left(\rho_t^i,\widetilde{\rho}_t^i\right) \leq e^{-\beta\kappa}\sum_{t=1}^{\infty}\gamma^t\zeta^t = \frac{\gamma\zeta}{1-\gamma\zeta}e^{-\beta\kappa}, \tag{32}$$

where $\zeta$ is defined in Assumption 2.1.

Let

$$\delta^j Q^i_{\boldsymbol{\pi}}(\mathbf{s}, \mathbf{a}) = \sup_{\mathbf{z}^j, \mathbf{z}^{-j}, z'^j} \left| Q^i_{\boldsymbol{\pi}}\left(\mathbf{z}^j, \mathbf{z}^{-j}\right) - Q^i_{\boldsymbol{\pi}}\left(z'^j, \mathbf{z}^{-j}\right) \right|,$$

and the MDP satisfies the condition of Lemma A.4, one can obtain

$$\sum_{j \in \mathcal{N}} e^{\beta d(i,j)} \delta^j \left(Q^i_{\boldsymbol{\pi}}(\mathbf{s}, \cdot)\right) \le \sum_{t=1}^{\infty} \gamma^t \sum_{j \in \mathcal{N}} e^{\beta d(i,j)} \delta^j P^i \le \sum_{t=1}^{\infty} \gamma^t \zeta^t = \frac{\gamma\zeta}{1 - \gamma\zeta}.$$

Secondly, building on Assumption 2.2 and Lemma A.4, we analyze $\left|V^i_{\boldsymbol{\pi}}(\mathbf{s}) - V^i_{\boldsymbol{\pi}}(\widetilde{\mathbf{s}})\right|$ can further obtain

$$
\begin{aligned}
&\left|V^i_{\boldsymbol{\pi}}(\mathbf{s}) - V^i_{\boldsymbol{\pi}}(\widetilde{\mathbf{s}})\right| \\
&= \left|\mathbb{E}_{\mathbf{a}\sim\boldsymbol{\pi}(\cdot|\mathbf{s})} Q^i_{\boldsymbol{\pi}}(\mathbf{s}, \mathbf{a}) - \mathbb{E}_{\mathbf{a}\sim\boldsymbol{\pi}(\cdot|\widetilde{\mathbf{s}})} Q^i_{\boldsymbol{\pi}}(\widetilde{\mathbf{s}}, \mathbf{a})\right| \\
&= \left|\mathbb{E}_{\mathbf{a}\sim\boldsymbol{\pi}(\cdot|\mathbf{s})} Q^i_{\boldsymbol{\pi}}(\mathbf{s}, \mathbf{a}) - \mathbb{E}_{\mathbf{a}\sim\boldsymbol{\pi}(\cdot|\widetilde{\mathbf{s}})} Q^i_{\boldsymbol{\pi}}(\mathbf{s}, \mathbf{a}) + \mathbb{E}_{\mathbf{a}\sim\boldsymbol{\pi}(\cdot|\widetilde{\mathbf{s}})} Q^i_{\boldsymbol{\pi}}(\mathbf{s}, \mathbf{a}) - \mathbb{E}_{\mathbf{a}\sim\boldsymbol{\pi}(\cdot|\widetilde{\mathbf{s}})} Q^i_{\boldsymbol{\pi}}(\widetilde{\mathbf{s}}, \mathbf{a})\right| \\
&\le \left|\mathbb{E}_{\mathbf{a}\sim\boldsymbol{\pi}(\cdot|\mathbf{s})} Q^i_{\boldsymbol{\pi}}(\mathbf{s}, \mathbf{a}) - \mathbb{E}_{\mathbf{a}\sim\boldsymbol{\pi}(\cdot|\widetilde{\mathbf{s}})} Q^i_{\boldsymbol{\pi}}(\mathbf{s}, \mathbf{a})\right| + \left|\mathbb{E}_{\mathbf{a}\sim\boldsymbol{\pi}(\cdot|\widetilde{\mathbf{s}})} Q^i_{\boldsymbol{\pi}}(\mathbf{s}, \mathbf{a}) - \mathbb{E}_{\mathbf{a}\sim\boldsymbol{\pi}(\cdot|\widetilde{\mathbf{s}})} Q^i_{\boldsymbol{\pi}}(\widetilde{\mathbf{s}}, \mathbf{a})\right| \\
&\le \sum_{j \in \mathcal{N}} D_{\mathrm{TV}}\left(\pi^j(\cdot \mid \mathbf{s}), \pi^j(\cdot \mid \widetilde{\mathbf{s}})\right) \delta^j Q^i_{\boldsymbol{\pi}}(\mathbf{s}, \mathbf{a}) + \frac{\gamma\zeta}{1 - \gamma\zeta} e^{-\beta\kappa} \\
&\le \xi e^{-\beta\kappa} \sum_{j \in \mathcal{N}} e^{-\beta d(j,i)} \delta^j Q^i_{\boldsymbol{\pi}}(\mathbf{s}, \mathbf{a}) + \frac{\gamma\zeta}{1 - \gamma\zeta} e^{-\beta\kappa} \\
&\le \frac{\gamma\zeta}{1 - \gamma\zeta} \xi e^{-\beta\kappa} + \frac{\gamma\zeta}{1 - \gamma\zeta} e^{-\beta\kappa} \\
&\le \frac{(1 + \xi)\gamma\zeta}{1 - \gamma\zeta} e^{-\beta\kappa}.
\end{aligned}
\tag{33}
$$

Then, bringing (32) and (33) into (31), we have

$$
\begin{aligned}
&\left|A^i_{\boldsymbol{\pi}}(\mathbf{s}, \mathbf{a}) - A^i_{\boldsymbol{\pi}}(\widetilde{\mathbf{s}}, \widetilde{\mathbf{a}})\right| \\
&\le \left|Q^i_{\boldsymbol{\pi}}(\mathbf{s}, \mathbf{a}) - Q^i_{\boldsymbol{\pi}}(\widetilde{\mathbf{s}}, \widetilde{\mathbf{a}})\right| + \left|V^i_{\boldsymbol{\pi}}(\mathbf{s}) - V^i_{\boldsymbol{\pi}}(\widetilde{\mathbf{s}})\right| \\
&\le \left|\mathbb{E}_{\mathbf{a}\sim\boldsymbol{\pi}(\cdot|\mathbf{s})} Q^i_{\boldsymbol{\pi}}(\mathbf{s}, \mathbf{a}) - \mathbb{E}_{\mathbf{a}\sim\boldsymbol{\pi}(\cdot|\widetilde{\mathbf{s}})} Q^i_{\boldsymbol{\pi}}(\mathbf{s}, \mathbf{a})\right| + 2\left|\mathbb{E}_{\mathbf{a}\sim\boldsymbol{\pi}(\cdot|\widetilde{\mathbf{s}})} Q^i_{\boldsymbol{\pi}}(\mathbf{s}, \mathbf{a}) - \mathbb{E}_{\mathbf{a}\sim\boldsymbol{\pi}(\cdot|\widetilde{\mathbf{s}})} Q^i_{\boldsymbol{\pi}}(\widetilde{\mathbf{s}}, \widetilde{\mathbf{a}})\right| \\
&\le \frac{\gamma\zeta}{1 - \gamma\zeta} \xi e^{-\beta\kappa} + \frac{2\gamma\zeta}{1 - \gamma\zeta} e^{-\beta\kappa} \\
&\le \frac{(2 + \xi)\gamma\zeta}{1 - \gamma\zeta} e^{-\beta\kappa}.
\end{aligned}
\tag{34}
$$

Finally, denoting $(\eta, \phi) = \left(\frac{(2+\xi)\gamma\zeta}{1-\gamma\zeta}, e^{-\beta}\right)$, we can obtain the Proposition 3.3. □

### C.4 The proof of Corollary 3.4

*Proof.* Firstly, according to Lamma 3.1 and Definition 3.2, the following result holds when each agent adopts a sequential update scheme to optimize policy, i.e., we have

$$
\begin{aligned}
&\left|L^{1:i}_{\boldsymbol{\pi}}\left(\bar{\boldsymbol{\pi}}^{1:i-1}, \bar{\pi}^i\right) - L^i_{\pi_\kappa}\left(\bar{\pi}^i_\kappa\right)\right| \\
&= \left|\mathbb{E}_{\mathbf{s}\sim\rho_{\boldsymbol{\pi}}, \mathbf{a}^{1:i}\sim\bar{\boldsymbol{\pi}}^{1:i}}\left[A^i_{\boldsymbol{\pi}}\left(\mathbf{s}, \mathbf{a}^{1:i-1}, a^i\right)\right] - \mathbb{E}_{\mathbf{s}_{\mathcal{N}^i_\kappa}\sim\rho_{\pi^i_\kappa}, a^i\sim\bar{\pi}^i}\left[A^i_{\pi^i_\kappa}\left(\mathbf{s}_{\mathcal{N}^i_\kappa}, a^i\right)\right]\right| \\
&= \left|\mathbb{E}_{\mathbf{s}\sim\rho_{\boldsymbol{\pi}}, \mathbf{a}^{1:i}\sim\bar{\boldsymbol{\pi}}^{1:i}}\left[A^i_{\boldsymbol{\pi}}\left(\mathbf{s}, \mathbf{a}^{1:i-1}, a^i\right)\right] - \mathbb{E}_{\mathbf{s}\sim\rho_{\boldsymbol{\pi}}, a^i\sim\bar{\pi}^i}\left[A^i_{\boldsymbol{\pi}}\left(\mathbf{s}, a^i\right)\right]\right. \\
&\quad \left. + \mathbb{E}_{\mathbf{s}\sim\rho_{\boldsymbol{\pi}}, a^i\sim\bar{\pi}^i}\left[A^i_{\boldsymbol{\pi}}\left(\mathbf{s}, a^i\right)\right] - \mathbb{E}_{\widetilde{\mathbf{s}}\sim\widetilde{\rho}_{\boldsymbol{\pi}}, a^i\sim\bar{\pi}^i}\left[A^i_{\boldsymbol{\pi}}\left(\widetilde{\mathbf{s}}, a^i\right)\right]\right| \\
&\le \left|\mathbb{E}_{\mathbf{s}\sim\rho_{\boldsymbol{\pi}}, \mathbf{a}^{1:i}\sim\bar{\boldsymbol{\pi}}^{1:i}}\left[A^i_{\boldsymbol{\pi}}\left(\mathbf{s}, \mathbf{a}^{1:i-1}, a^i\right)\right] - \mathbb{E}_{\mathbf{s}\sim\rho_{\boldsymbol{\pi}}, a^i\sim\bar{\pi}^i}\left[A^i_{\boldsymbol{\pi}}\left(\mathbf{s}, a^i\right)\right]\right| \\
&\quad + \left|\mathbb{E}_{\mathbf{s}\sim\rho_{\boldsymbol{\pi}}, a^i\sim\bar{\pi}^i}\left[A^i_{\boldsymbol{\pi}}\left(\mathbf{s}, a^i\right)\right] - \mathbb{E}_{\widetilde{\mathbf{s}}\sim\widetilde{\rho}_{\boldsymbol{\pi}}, a^i\sim\bar{\pi}^i}\left[A^i_{\boldsymbol{\pi}}\left(\widetilde{\mathbf{s}}, a^i\right)\right]\right|.
\end{aligned}
\tag{35}
$$

Then, based on Assumptions 2.2, we have

$$
\begin{aligned}
&\left|\mathbb{E}_{\mathbf{s}\sim\boldsymbol{\rho}_{\boldsymbol{\pi}},\mathbf{a}^{1:i}\sim\bar{\boldsymbol{\pi}}^{1:i}}\left[A_{\boldsymbol{\pi}}^i\left(\mathbf{s},\mathbf{a}^{1:i-1},\mathrm{a}^i\right)\right]-\mathbb{E}_{\mathbf{s}\sim\boldsymbol{\rho}_{\boldsymbol{\pi}},\mathrm{a}^i\sim\bar{\pi}^i}\left[A_{\boldsymbol{\pi}}^i\left(\mathbf{s},\mathrm{a}^i\right)\right]\right| \\
&=\left|\mathbb{E}_{\mathbf{s}\sim\boldsymbol{\rho}_{\boldsymbol{\pi}},\mathrm{a}^i\sim\bar{\pi}^i}\left[\sum_{h=1}^{i-1}\left(\bar{\pi}^h-\pi^h\right)A_{\boldsymbol{\pi}}^i\left(\mathbf{s},\mathrm{a}^i\right)\right]\right| \\
&\leq\mathbb{E}_{\mathbf{s}\sim\boldsymbol{\rho}_{\boldsymbol{\pi}},\mathrm{a}^i\sim\bar{\pi}^i}\left[\sum_{h=1}^{i-1}\left|\bar{\pi}^h-\pi^h\right|\left|A_{\boldsymbol{\pi}}^i\left(\mathbf{s},\mathrm{a}^i\right)\right|\right] \\
&\leq\mathbb{E}_{\mathbf{s}\sim\boldsymbol{\rho}_{\boldsymbol{\pi}},\mathrm{a}^i\sim\bar{\pi}^i}\left[M^i\sum_{h=1}^{i-1}\left|\bar{\pi}^h-\pi^h\right|\right]\qquad M^i=\max_{\pi^i}\left|A_{\boldsymbol{\pi}}^i\left(\mathbf{s},\mathrm{a}^i\right)\right| \\
&\leq\frac{M^i}{1-\gamma}\sum_{h=1}^{i-1}\max_{\mathrm{s}}D_{\mathrm{TV}}(\bar{\pi}^h,\pi^h) \\
&\leq\frac{M^i\xi}{1-\gamma}e^{-\beta\kappa}.
\end{aligned}
\tag{36}
$$

According to (34), we have

$$
\begin{aligned}
&\left|\mathbb{E}_{\mathbf{s}\sim\boldsymbol{\rho}_{\boldsymbol{\pi}},\mathrm{a}^i\sim\bar{\pi}^i}\left[A_{\boldsymbol{\pi}}^i\left(\mathbf{s},\mathrm{a}^i\right)\right]-\mathbb{E}_{\widetilde{\mathbf{s}}\sim\widetilde{\boldsymbol{\rho}}_{\boldsymbol{\pi}},\mathrm{a}^i\sim\bar{\pi}^i}\left[A_{\boldsymbol{\pi}}^i\left(\widetilde{\mathbf{s}},\mathrm{a}^i\right)\right]\right| \\
&\leq\mathbb{E}\left|\left[A_{\boldsymbol{\pi}}^i(\mathbf{s},\mathbf{a})-A_{\boldsymbol{\pi}}^i(\widetilde{\mathbf{s}},\widetilde{\mathbf{a}})\right]\right| \\
&\leq\frac{(2+\xi)\gamma\zeta}{1-\gamma\zeta}e^{-\beta\kappa}.
\end{aligned}
\tag{37}
$$

Then, bringing (36) and (37) into (35), we have

$$
\begin{aligned}
&\left|L_{\boldsymbol{\pi}}^{1:i}\left(\bar{\boldsymbol{\pi}}^{1:i-1},\bar{\pi}^i\right)-L_{\boldsymbol{\pi}}^i\left(\bar{\pi}_\kappa^i\right)\right| \\
&\leq\left|\mathbb{E}_{\mathbf{s}\sim\boldsymbol{\rho}_{\boldsymbol{\pi}},\mathbf{a}^{1:i}\sim\bar{\boldsymbol{\pi}}^{1:i}}\left[A_{\boldsymbol{\pi}}^i\left(\mathbf{s},\mathbf{a}^{1:i-1},\mathrm{a}^i\right)\right]-\mathbb{E}_{\mathbf{s}\sim\boldsymbol{\rho}_{\boldsymbol{\pi}},\mathrm{a}^i\sim\bar{\pi}^i}\left[A_{\boldsymbol{\pi}}^i\left(\mathbf{s},\mathrm{a}^i\right)\right]\right| \\
&\quad+\left|\mathbb{E}_{\mathbf{s}\sim\boldsymbol{\rho}_{\boldsymbol{\pi}},\mathrm{a}^i\sim\bar{\pi}^i}\left[A_{\boldsymbol{\pi}}^i\left(\mathbf{s},\mathrm{a}^i\right)\right]-\mathbb{E}_{\widetilde{\mathbf{s}}\sim\widetilde{\boldsymbol{\rho}}_{\boldsymbol{\pi}},\mathrm{a}^i\sim\bar{\pi}^i}\left[A_{\boldsymbol{\pi}}^i\left(\widetilde{\mathbf{s}},\mathrm{a}^i\right)\right]\right| \\
&\leq\frac{M^i\xi}{1-\gamma}e^{-\beta\kappa}+\frac{(2+\xi)\gamma\zeta}{1-\gamma\zeta}e^{-\beta\kappa} \\
&\leq\left(\frac{M^i\xi}{1-\gamma}+\frac{(2+\xi)\gamma\zeta}{1-\gamma\zeta}\right)e^{-\beta\kappa}.
\end{aligned}
$$

Finally, denoting $(\eta',\phi)=\left(\frac{M^i\xi}{1-\gamma}+\frac{(2+\xi)\gamma\zeta}{1-\gamma\zeta},e^{-\beta}\right)$, we can obtain the Corollary 3.4. $\qquad\square$

### C.5   The proof of Proposition 3.5

*Proof.* From (12), we can obtain $-\eta'\phi^\kappa\leq L_{\boldsymbol{\pi}}^{1:i}\left(\bar{\boldsymbol{\pi}}^{1:i-1},\bar{\pi}^i\right)-L_{\pi_\kappa^i}^i\left(\bar{\pi}_\kappa^i\right)\leq\eta'\phi^\kappa$. By the trust region theorem in Theorem 1 from [17], we have

$$
\begin{aligned}
J\left(\bar{\boldsymbol{\pi}}\right)-J\left(\boldsymbol{\pi}\right)&\geq\mathbb{E}_{\mathbf{s}\sim\boldsymbol{\rho}_{\boldsymbol{\pi}},\mathbf{a}\sim\bar{\boldsymbol{\pi}}}\left[A_{\boldsymbol{\pi}}(\mathbf{s},\mathbf{a})\right]-\nu D_{\mathrm{KL}}^{\max}\left(\boldsymbol{\pi},\bar{\boldsymbol{\pi}}\right) \\
&\geq\mathbb{E}_{\mathbf{s}\sim\boldsymbol{\rho}_{\boldsymbol{\pi}},\mathbf{a}\sim\bar{\boldsymbol{\pi}}}\left[A_{\boldsymbol{\pi}}(\mathbf{s},\mathbf{a})\right]-\sum_{i=1}^n\nu D_{\mathrm{KL}}^{\max}\left(\pi^i,\bar{\pi}^i\right) \\
&=\sum_{i=1}^n\mathbb{E}_{\mathbf{s}\sim\boldsymbol{\rho}_{\boldsymbol{\pi}},\mathbf{a}^{1:i}\sim\boldsymbol{\pi}^{1:i}}\left[A_{\boldsymbol{\pi}}^i\left(\mathbf{s},\mathbf{a}^{1:i-1},a^i\right)\right]-\sum_{i=1}^n\nu D_{\mathrm{KL}}^{\max}\left(\pi^i,\bar{\pi}^i\right) \\
&=\sum_{i=1}^n\left(L_{\boldsymbol{\pi}}^{1:i}\left(\bar{\boldsymbol{\pi}}^{1:i-1},\bar{\pi}^i\right)-\nu D_{\mathrm{KL}}^{\max}\left(\pi^i,\bar{\pi}^i\right)\right) \\
&\geq\sum_{i=1}^n\left(L_{\pi_\kappa^i}^i\left(\hat{\pi}_\kappa^i\right)-\eta'\phi^\kappa-\nu D_{\mathrm{KL}}^{\max}\left(\pi^i|\hat{\pi}^i\right)\right) \\
&\geq\sum_{i=1}^n\left(L_{\pi_\kappa^i}^i\left(\hat{\pi}_\kappa^i\right)-\eta'\phi^\kappa-\nu_\kappa^i D_{\mathrm{KL}}^{\max}\left(\pi_\kappa^i|\hat{\pi}_\kappa^i\right)\right).
\end{aligned}
$$

Then, when each agent sequentially solves the following optimization problem:

$$\bar{\pi}_\kappa^i = \arg\max_{\hat{\pi}_\kappa^i} \left( L_{\pi_\kappa}^i \left( \hat{\pi}_\kappa^i \right) - \eta' \phi^\kappa - \nu_\kappa^i D_{\mathrm{KL}}^{\max} \left( \pi_\kappa^i | \hat{\pi}_\kappa^i \right) \right),$$

where $(\eta', \phi) = \left( \frac{M^i \xi}{1-\gamma} + \frac{(2+\xi)\gamma\zeta}{1-\gamma\zeta}, e^{-\beta} \right)$, $\nu_\kappa^i = \frac{2\gamma \max_{s_{\mathcal{N}_\kappa^i}, a^i} \left| A_{\pi_\kappa}^i (s_{\mathcal{N}_\kappa^i}, a^i) \right|}{(1-\gamma)^2}$, and $D_{\mathrm{KL}}^{\max} \left( \pi_\kappa^i | \hat{\pi}_\kappa^i \right) = \max_{s_{\mathcal{N}_\kappa^i}} D_{\mathrm{KL}} \left( \pi^i(\cdot \mid s_{\mathcal{N}_\kappa^i}), \hat{\pi}^i(\cdot \mid s_{\mathcal{N}_\kappa^i}) \right)$, we have $J(\bar{\pi}) - J(\pi) \geq \sum_{i=1}^n \left( L_{\pi_\kappa}^i \left( \hat{\pi}_\kappa^i \right) - \eta' \phi^\kappa - \nu_\kappa^i D_{\mathrm{KL}}^{\max} \left( \pi_\kappa^i | \hat{\pi}_\kappa^i \right) \right).$ □

### C.6 The proof of Corollary 3.6

*Proof.* Firstly, by generalizing the result about the return in (14), one can derive how the expected costs change when the agents update their policies. Inspired by [10], we provide the following lemma.

*Lemma* C.3. *Let $\pi$ and $\bar{\pi}$ be joint policies. Let $i \in \mathcal{N}$ be an agent, and $j \in \{1, \ldots, m^i\}$ be an index of one of its costs. The following inequality holds*

$$J_j^i(\bar{\pi}) \leq J_j^i(\pi) + L_{j,\pi}^i \left( \bar{\pi}^i \right) + \nu_j^i \sum_{h=1}^i D_{\mathrm{KL}}^{\max} \left( \pi^h, \bar{\pi}^h \right).$$

*where $L_{j,\pi}^i(\bar{\pi}^i) = \mathbb{E}_{s \sim \rho_\pi, a^i \sim \bar{\pi}^i} \left[ A_{j,\pi}^i (s, a^i) \right]$, $\nu_j^i = \frac{2\gamma \max_{s, a^i} \left| A_{j,\pi}^i (s, a^i) \right|}{(1-\gamma)^2}$.*

*Proof.* From the upper bound version of Theorem 1 of [17] applied to joint policies $\bar{\pi}$ and $\pi$, we conclude that

$$J_j^i(\bar{\pi}) \leq J_j^i(\pi) + \mathbb{E}_{s \sim \rho_\pi, a^{1:i} \sim \bar{\pi}^{1:i}} \left[ A_{j,\pi}^i (s, a^i) \right] + \frac{4\alpha^2 \gamma \max_{s, a^i} \left| A_{j,\pi}^i (s, a^i) \right|}{(1-\gamma)^2},$$

where $\alpha = D_{\mathrm{TV}}^{\max}(\pi^{1:i}, \bar{\pi}^{1:i}) = \max_s D_{\mathrm{TV}}(\pi^{1:i}(\cdot \mid s), \bar{\pi}^{1:i}(\cdot \mid s))$.

Then, using Pinsker's inequality $D_{\mathrm{TV}}(p, q)^2 \leq D_{\mathrm{KL}}(p, q)/2$, we obtain

$$J_j^i(\bar{\pi}) \leq J_j^i(\pi) + \mathbb{E}_{s \sim \rho_\pi, a^{1:i} \sim \bar{\pi}^{1:i}} \left[ A_{j,\pi}^i (s, a^i) \right] + \frac{2\gamma \max_{s, a^i} \left| A_{j,\pi}^i (s, a^i) \right|}{(1-\gamma)^2} D_{\mathrm{TV}}^{\max}(\pi^{1:i}, \bar{\pi}^{1:i}),$$

where $D_{\mathrm{KL}}^{\max}(\pi^{1:i}, \bar{\pi}^{1:i}) = \max_s D_{\mathrm{KL}}(\pi^{1:i}(\cdot \mid s), \bar{\pi}^{1:i}(\cdot \mid s))$.

Notice that we have $\mathbb{E}_{s \sim \rho_\pi, a^{1:i} \sim \bar{\pi}^{1:i}} \left[ A_{j,\pi}^i (s, a^i) \right] = \mathbb{E}_{s \sim \rho_\pi, a^i \sim \bar{\pi}^i} \left[ A_{j,\pi}^i (s, a^i) \right]$ as the action of agents other that $i$ do not change the value of the variable inside of the expectation. Furthermore,

$$\begin{aligned} D_{\mathrm{KL}}^{\max}(\pi^{1:i}, \bar{\pi}^{1:i}) &= \max_s D_{\mathrm{KL}}(\pi^{1:i}(\cdot \mid s), \bar{\pi}^{1:i}(\cdot \mid s)) \\ &\leq \max_s \left( \sum_{h=1}^i D_{\mathrm{KL}} \left( \pi^h(\cdot \mid s), \bar{\pi}^h(\cdot \mid s) \right) \right) \\ &\leq \sum_{h=1}^i \max_s \left( D_{\mathrm{KL}} \left( \pi^h(\cdot \mid s), \bar{\pi}^h(\cdot \mid s) \right) \right) \\ &= \sum_{h=1}^i D_{\mathrm{KL}}^{\max} \left( \pi^h, \bar{\pi}^h \right). \end{aligned}$$

Setting $\nu_j^i = \frac{2\gamma \max_{s, a^i} \left| A_{j,\pi}^i (s, a^i) \right|}{(1-\gamma)^2}$, we finally obtain

$$J_j^i(\bar{\pi}) \leq J_j^i(\pi) + L_{j,\pi}^i \left( \bar{\pi}^i \right) + \nu_j^i \sum_{h=1}^{i-1} D_{\mathrm{KL}}^{\max} \left( \pi^h, \bar{\pi}^h \right).$$

□

Secondly, from (12), we can obtain $-\eta'\phi^\kappa \le L_{\boldsymbol{\pi}}^{1:i}\left(\bar{\boldsymbol{\pi}}^{1:i-1}, \bar{\pi}^i\right) - L_{\pi_\kappa^i}^i\left(\bar{\pi}_\kappa^i\right) \le \eta'\phi^\kappa$. By generalizing the result, we can obtain $-\eta''\phi^\kappa \le L_{j,\boldsymbol{\pi}}^{1:i}\left(\bar{\boldsymbol{\pi}}^{1:i-1}, \bar{\pi}^i\right) - L_{j,\pi_\kappa^i}^i\left(\bar{\pi}_\kappa^i\right) \le \eta''\phi^\kappa$. Further, we can derive the upper bounds about surrogate cost

$$J_j^i(\bar{\boldsymbol{\pi}}) \le J_j^i(\boldsymbol{\pi}) + L_{j,\boldsymbol{\pi}}^i(\bar{\pi}^i) + \nu_j^i \sum_{h=1}^n D_{KL}^{max}(\pi^h, \widetilde{\pi}^h)$$

$$\le J_j^i(\boldsymbol{\pi}) + L_{j,\pi_\kappa^i}^i\left(\bar{\pi}_\kappa^i\right) + \eta''\phi^\kappa + \nu_{j,\kappa}^i \sum_{h=1}^i D_{KL}^{\max}\left(\pi_\kappa^h, \bar{\pi}_\kappa^h\right).$$

where $L_{j,\pi_\kappa^i}^i(\bar{\pi}_\kappa^i) = \mathbb{E}_{s_{\pi_\kappa^i} \sim \rho_{\pi_\kappa^i}, a^i \sim \bar{\pi}^i}\left[A_{j,\pi_\kappa^i}^i\left(s_{\mathcal{N}_\kappa^i}, a^i\right)\right]$, $(\eta'', \phi) = \left(\frac{M_j\xi}{1-\gamma} + \frac{(2+\xi)\gamma\zeta}{1-\gamma\zeta}, e^{-\beta}\right)$, $\nu_{j,\kappa}^i = \frac{2\gamma \max_{s_{\mathcal{N}_\kappa^i}, a^i}\left|A_{j,\pi_\kappa^i}^i\left(s_{\mathcal{N}_\kappa^i}, a^i\right)\right|}{(1-\gamma)^2}$, and $M_j$ is a constant. $\qquad\square$

### C.7 The proof of Theorem 3.7

*Proof.* Based on the conclusions in Proposition 3.5 and Corollary 3.6, we can derive that in order to realize reward performance improvement and satisfy safety constraints, agents have to sequentially maximize their surrogate returns and ensure that their surrogate costs stay below the corresponding safety thresholds. Meanwhile, they have to constrain the policy search to small local neighborhoods (w.rt, max-KL distance). Therefore, the size of KL constraint in Equation (16) should be set as

$$\delta_\kappa^i = \min\left\{ \min_{h \le i-1} \min_{1 \le j \le m^h} \frac{c_j^h - J_j^h(\boldsymbol{\pi}) - L_{j,\pi_\kappa^h}^i\left(\bar{\pi}_\kappa^h\right) - \eta''\phi^\kappa - \nu_{j,\kappa}^h \sum_{l=1}^{i-1} D_{KL}^{\max}\left(\pi_\kappa^l, \bar{\pi}_\kappa^l\right)}{\nu_{j,\kappa}^h}, \right.$$
$$\left. \min_{h \ge i+1} \min_{1 \le j \le m^h} \frac{c_j^h - J_j^h(\boldsymbol{\pi}) - \nu_{j,\kappa}^h \sum_{l=1}^{i-1} D_{KL}^{\max}\left(\pi_\kappa^l, \bar{\pi}_\kappa^l\right)}{\nu_{j,\kappa}^h} \right\},$$

(38)

where $h \in \mathcal{N}_\kappa^i$ is the $\kappa$-hop neighbors of agent $i$, and $j \in \{1, \dots, m^h\}$ is its cost index.

Note that $\delta_\kappa^1$ is guaranteed to be non-negative if $\boldsymbol{\pi}$ satisfies safety constraints; that is because then $c_j^h \ge J_j^h(\boldsymbol{\pi})$ for all $h \in \mathcal{N}$, and $j \in \{1, \dots, m^i\}$, and the set $\{h \mid h < i\}$ is empty.

This formula for $\delta_\kappa^i$, combined with Lemma 3.1, assures that the policies $\pi_\kappa^i$ within $\delta_\kappa^i$ max-KL distance from $\pi_\kappa^i$ will not violate other agents' safety constraints, as long as the base joint policy $\boldsymbol{\pi}$ did not violate them (which assures $\delta_\kappa^1 \ge 0$). To see this, for every $h = 1, \dots, i-1$, and $j = 1, \dots, m^h$, we have

$$D_{KL}^{\max}\left(\pi_\kappa^i, \bar{\pi}_\kappa^i\right) \le \delta_\kappa^i \le \frac{c_j^h - J_j^h(\boldsymbol{\pi}) - L_{j,\pi_\kappa^h}^i\left(\bar{\pi}_\kappa^h\right) - \eta''\phi^\kappa - \nu_{j,\kappa}^h \sum_{l=1}^{i-1} D_{KL}^{\max}\left(\pi_\kappa^l, \bar{\pi}_\kappa^l\right)}{\nu_{j,\kappa}^h},$$

which implies

$$J_j^h(\boldsymbol{\pi}) + L_{j,\pi_\kappa^h}^i\left(\bar{\pi}_\kappa^h\right) + \eta''\phi^\kappa + \nu_{j,\kappa}^h \sum_{l=1}^{i-1} D_{KL}^{\max}\left(\pi_\kappa^l, \bar{\pi}_\kappa^l\right) + \nu_{j,\kappa}^h D_{KL}^{\max}\left(\pi_\kappa^i, \bar{\pi}_\kappa^i\right) \le c_j^h. \quad (39)$$

By Corollary 3.6, the left-hand side of the inequality (39) is an upper bound of $J_j^h\left(\bar{\boldsymbol{\pi}}^{1:i-1}, \pi^i\right)$, which implies that the update of agent $i$ does not violate the constraint of $J_j^h$. The fact that the constraints of $J_j^h$ for $h \ge i+1$ are not violated, i.e.,

$$J_j^h(\boldsymbol{\pi}) + \nu_{j,\kappa}^h \sum_{l=1}^{i-1} D_{KL}^{\max}\left(\pi_\kappa^l, \bar{\pi}_\kappa^l\right) + \nu_{j,\kappa}^h D_{KL}^{\max}\left(\pi_\kappa^i, \bar{\pi}_\kappa^i\right) \le c_j^h.$$

Therefore, let $(\eta', \phi) = \left(\frac{M^i\xi}{1-\gamma} + \frac{(2+\xi)\gamma\zeta}{1-\gamma\zeta}, e^{-\beta}\right)$, $(\eta'', \phi) = \left(\frac{M_j\xi}{1-\gamma} + \frac{(2+\xi)\gamma\zeta}{1-\gamma\zeta}, e^{-\beta}\right)$,
$\nu_\kappa^i = \frac{2\gamma \max_{s_{\mathcal{N}_\kappa^i}, a^i}\left|A_{\pi_\kappa^i}^i\left(s_{\mathcal{N}_\kappa^i}, a^i\right)\right|}{(1-\gamma)^2}$, $\nu_{j,\kappa}^i = \frac{2\gamma \max_{s_{\mathcal{N}_\kappa^i}, a^i}\left|A_{j,\pi_\kappa^i}^i\left(s_{\mathcal{N}_\kappa^i}, a^i\right)\right|}{(1-\gamma)^2}$, $\delta_\kappa^i =$

$$\min\left\{\min_{h\leq i-1}\min_{1\leq j\leq m^h}\frac{\Xi_j^h - L_{j,\pi_\kappa^h}^h\left(\bar{\pi}_\kappa^h\right)-\eta''\phi^\kappa}{\nu_{j,\kappa}^i},\min_{h\geq i+1}\min_{1\leq j\leq m^h}\frac{\Xi_j^h}{\nu_{j,\kappa}^i}\right\},\qquad \Xi_j^h \quad =$$

$c_j^h - J_j^h\left(\pi_\kappa^h\right) - \nu_{j,\kappa}^h\sum_{l=1}^{i-1}D_{\mathrm{KL}}^{\max}\left(\pi_\kappa^l,\hat{\pi}_\kappa^l\right)$, when the policy is updated by following a sequential update scheme, that is, each agent sequentially solves the following optimization problem:

$$\bar{\pi}_\kappa^i = \underset{\hat{\pi}_\kappa^i\in\bar{\Pi}_\kappa^i}{\arg\max}\left(L_{\pi_\kappa^i}^i\left(\hat{\pi}_\kappa^i\right)-\eta'\phi^\kappa-\nu_\kappa^iD_{\mathrm{KL}}^{\max}\left(\pi_\kappa^i|\hat{\pi}_\kappa^i\right)\right),$$

$$s.t.\left\{\hat{\pi}_\kappa^i\in\bar{\Pi}_\kappa^i\mid D_{\mathrm{KL}}^{\max}\left(\pi_\kappa^i,\hat{\pi}_\kappa^i\right)\leq\delta_\kappa^i,\text{and}\right.$$

$$\left.J_j^i\left(\pi_\kappa\right)+L_{j,\pi_\kappa^i}^i\left(\hat{\pi}_\kappa^i\right)+\eta''\phi^\kappa+\nu_{j,\kappa}^iD_{\mathrm{KL}}^{\max}\left(\pi_\kappa^i,\hat{\pi}_\kappa^i\right)\leq c_j^i-\nu_{j,\kappa}^i\sum_{h=1}^{i-1}D_{\mathrm{KL}}^{\max}\left(\pi_\kappa^h,\hat{\pi}_\kappa^h\right)\right\},$$

the joint policy $\boldsymbol{\pi}$ has the monotonic improvement property, $J\left(\bar{\boldsymbol{\pi}}\right)\geq J\left(\boldsymbol{\pi}\right)$, as well as it satisfies the safety constraints, $J_j^i\left(\bar{\boldsymbol{\pi}}\right)\leq c_j^i$, for any agent $i\in\mathcal{N}$ and its cost index $j\in\{1,\ldots,m^i\}$. $\qquad\square$

### C.8 Algorithm

In this subsection, we provide the main pseudocode for Scalable MAPPO-Lagrangian (Scal-MAPPO-L), as outlined in Algorithm 1.

---

**Algorithm 1** Scalable MAPPO-Lagrangian

---

**Input:** Stepsizes $\alpha_\theta,\alpha_\lambda$, batch size $B$, number of agents $n$, episodes $Z$, steps per episode $T$, discount factor $\gamma$, parameter $\kappa$.
**Initialize:** Actor networks $\theta_{\kappa,0}^1,\ldots,\theta_{\kappa,0}^n$, V-value network $\chi_{\kappa,0}^1,\ldots,\chi_{\kappa,0}^n$, V-cost networks $\left\{\phi_{j,0}^i\right\}_{1\leq j\leq m^i}^{i\in\mathcal{N}},\forall i\in\mathcal{N},j\in1,\ldots,m^i$, Replay buffer $\mathcal{B}$.

1: **for** $z=0,1,\ldots,Z-1$ **do**
2:      Collect a set of trajectories by running the policies $\boldsymbol{\pi}_{\theta_\kappa^1},\ldots,\boldsymbol{\pi}_{\theta_\kappa^n}$.
3:      Push transitions $\left\{\left(o_t^i,a_t^i,o_{t+1}^i,r_t^i\right),\forall i\in\mathcal{N},t\in T\right\}$ into $\mathcal{B}$.
4:      Sample a random minibatch of $B$ transitions from $\mathcal{B}$.
5:      **for** $i=1:n$ **do**
6:          Initialize a policy parameter $\theta_{\kappa,0}^i$ and Lagrangian multipliers $\lambda_j^i,\forall i\in\mathcal{N},j\in1,\ldots,m^i$.
7:          Compute advantage function $\hat{A}^i(\mathbf{s},\mathrm{a}^i)$ and cost advantage functions $\hat{A}_j^i(\mathbf{s},\mathrm{a}^i)$.
8:          Compute the parameters $\eta',\eta'',\phi,\nu_\kappa^i$ and $\nu_{j,\kappa}^i,\forall j\in\{1,\ldots,m^i\}$.
9:          Compute the radius of the KL-constraint $\delta_\kappa^i$.
10:        Compute the advantage function in (18).
11:        Update policy according to (20).
12:        Update V-value network and V-cost networks.
13:      **end for**
14: **end for**

---

The algorithm has a simple idea that each agent independently optimizes the surrogate objective, which only depends on its action and the state of its $\kappa$-hop neighbors for each agent. In the actual execution, we adopt the surrogate objective (20) instead of (16). It actually uses some approximations for the decentralized surrogate objective, the same as the MAPPO-L [10]. Most of these approximations are traditional practices in RL, yet they may make it impossible for the practical algorithm to rigorously maintain the theoretical guarantees in Theorem 3.7. However, we need to argue that we should go one step further and provide a decentralized surrogate for decentralized learning with a convergence guarantee. We believe and expect that a better practical method can be found based on this objective in future work.

# D   Supplementary materials for Section 4

## D.1   Additional experimental results

In this paper, we compare the algorithm of our proposed (i.e., Scal-MAPPO-L in Algorithm 1) against other PPO family algorithms on several safe MARL tasks to evaluate their performance. Here, we provide some additional experimental results, which are illustrated in Figures 3-4.

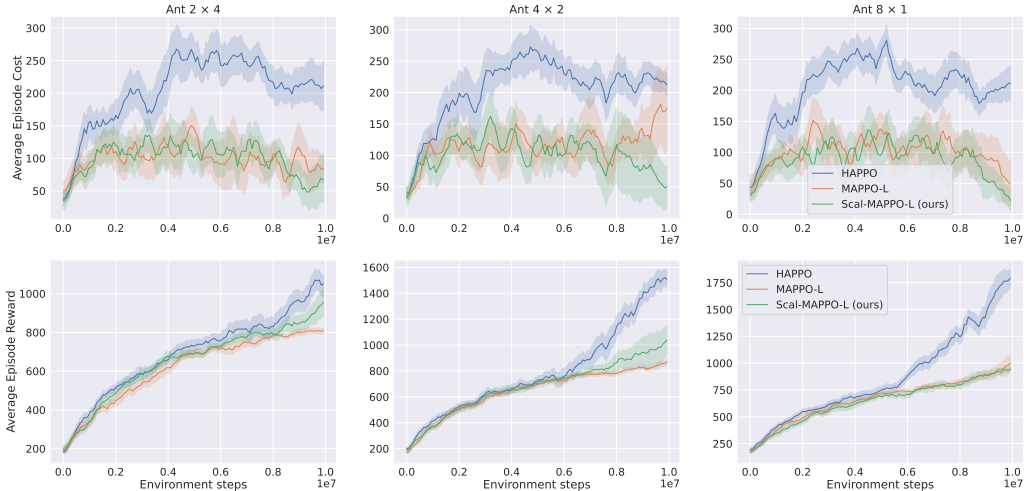

Figure 3: Performance comparisons in terms of cost and reward on three Safe Ant-v2 tasks. Each column subfigure represents a different task, and we plot the cost curves (the lower the better) in the upper row and the reward curves (the higher the better) in the bottom row for each task.

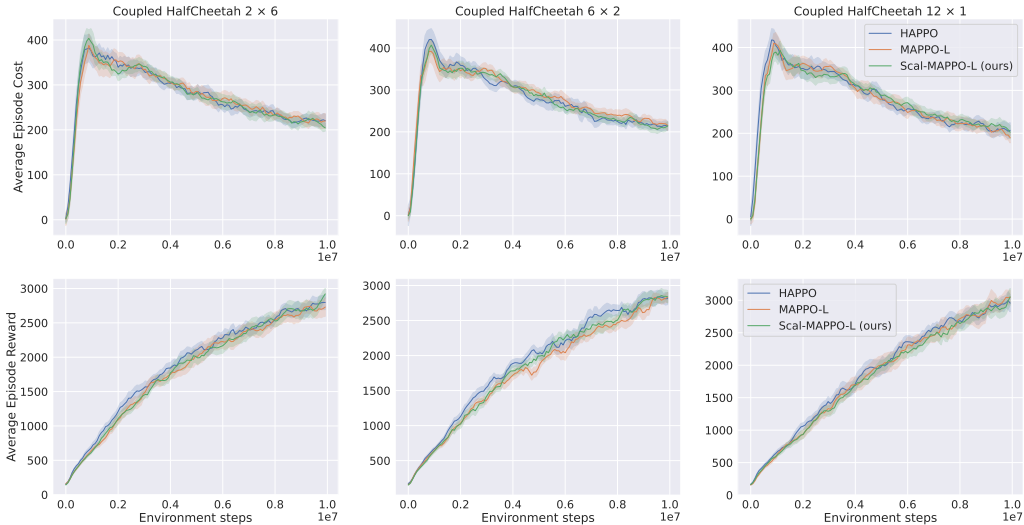

Figure 4: Performance comparisons in terms of cost and reward on three Safe Coupled HalfCheetah tasks.

*Remark* D.1.  It is worth pointing out that, in our code, unlike the original, the global state consists of a patchwork of each agent's ID and the $\kappa$-hop information rather than a long state vector. This is the main reason of the difference in performance from the original paper. As we consider decentralized learning, the agents in the experiments do not use parameter-sharing. In all experiments, the network architectures and common hyperparameters of our algorithm and MAPPO-L are the same for a fair

comparison. All reported results are averaged over three or more random seeds, and the curves are smooth over time.

### D.2  The computer resources and computational complexity

**Computation resources**: We executed our code on a computer with NVIDIA GeForce RTX 4090 (GPU) and Intel Core i9-13900K (CPU).

**Computational complexity**: The computational complexity of Scal-MAPPO-L (ours) is O(TNMHP), where T denotes the number of steps, N denotes the number of agents, M denotes the number of constraints, H denotes the number of PPO-Epoch, and P denotes the number of policy parameters.

Besides, we test the running time of Scal-MAPPO-L on Safe Manyagent Ant $6 \times 1$, Safe Ant $8 \times 1$, and Safe Coupled HalfCheetah $12 \times 1$. The running steps are $1 \times 10^7$ in each environment. When the parameter $\kappa$ is maximized, the algorithm's average wall-clock times are $8.43h$, $9.28h$, and $11.65h$, respectively. It is worth noting that the wall-clock times do not significantly down when $\kappa$ gradually decreases. This may be due to the fact that we have yet to consider the process of sending and receiving information realistically. However, based on the successful research conducted in the field of communication [51, 52], it is evident that algorithms requiring less communication undoubtedly have an advantage in terms of reducing communication burden and enhancing applicability.

## E  The discussion of limitations and impacts

### E.1  Limitations

This paper is centered on theoretical analysis and also contains practical algorithms and simulation verification. The main results in the paper characterize the proposed method's performance in terms of safety constraints and joint policy improvement. Below, we discuss the limitations of the proposed approach for both theory and experiment aspects as follows:

1) Our theoretical results are based on the two assumptions about the spatial decay of correlation for the dynamics and the policies in Assumption 2.1 and Assumption 2.2. Our conclusions may be useless when such assumptions do not hold, e.g., the decisions of each agent are non-negligibly related to the decisions of all other agents. However, fortunately, existing works [46, 47] have shown that many real-world situations satisfy both assumptions, so our study is still important and meaningful.

2) Our experiments show that Scal-MAPPO-L, with communication between a small number of neighbors, outperforms MAPPO-L in some cases, which we would like to see because it implies fewer communication requirements. However, we have yet to develop an equilibrium relationship between the amount of communication and the performance, which we will focus on next.

### E.2  Broader impacts

This paper presents work that aims to advance the field of RL, especially safe MARL. Our work has many positive societal impacts, such as providing a theoretical foundation for scalable Safe MARL, none of which we feel must be specifically highlighted. There are no negative societal impacts on our work.

